# Shapes analysis for time series

**Thibaut Germain**[1]*
Centre Borelli, ENS Paris-Saclay
4 av. des sciences, 91190

**Samuel Gruffaz**[1]
Centre Borelli, ENS Paris-Saclay
4 av. des sciences, 91190

**Charles Truong**[1]
Centre Borelli, ENS Paris-Saclay
4 av. des sciences, 91190

**Laurent Oudre**[1]
Centre Borelli, ENS Paris-Saclay
4 av. des sciences, 91190

**Alain Durmus**
CMAP, CNRS, Ecole polytechnique
Institut Polytechnique de Paris
91120 Palaiseau, France

## Abstract

Analyzing inter-individual variability of physiological functions is particularly appealing in medical and biological contexts to describe or quantify health conditions. Such analysis can be done by comparing individuals to a reference one with time series as biomedical data. This paper introduces an unsupervised representation learning (URL) algorithm for time series tailored to inter-individual studies. The idea is to represent time series as deformations of a reference time series. The deformations are diffeomorphisms parameterized and learned by our method called TS-LDDMM. Once the deformations and the reference time series are learned, the vector representations of individual time series are given by the parametrization of their corresponding deformation. At the crossroads between URL for time series and shape analysis, the proposed algorithm handles irregularly sampled multivariate time series of variable lengths and provides shape-based representations of temporal data. In this work, we establish a representation theorem for the graph of a time series and derive its consequences on the LDDMM framework. We showcase the advantages of our representation compared to existing methods using synthetic data and real-world examples motivated by biomedical applications.

## 1 Introduction

Our goal is to analyze the inter-individual variability within a time series dataset, an approach of significant interest in physiological contexts [25, 58, 4, 21]. Specifically, we aim to develop an unsupervised feature representation method that encodes the specificities of individual time series in comparison to a reference time series. In physiology, examining the various "shapes" in a time series related to biological phenomena and their variations due to individual differences or pathological conditions is common. However, the term "shape" lacks a precise definition and is more intuitively understood as the silhouette of a pattern in a time series. In this paper, we refer to the shape of a time series as the graph of this signal.

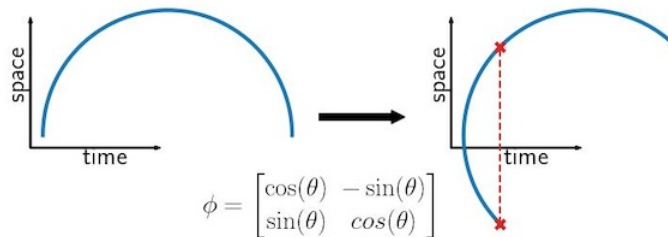

$$\phi = \begin{bmatrix} \cos(\theta) & -\sin(\theta) \\ \sin(\theta) & cos(\theta) \end{bmatrix}$$

Figure 1: A time series' graph $\mathsf{G} = \{(t, s(t)) : \ t \in \mathsf{I}\}$ can lose its structure after applying a general diffeomorphism $\phi.\mathsf{G}$: a time value can be related to two values on the space axis.

Although community structures with representatives can be learned in an unsupervised manner [55, 39] using contrastive loss [20, 54, 39] or similarity measures [2, 21, 45, 62], the study of inter-individual variability of shapes within a cluster [42, 51] remains an open problem in unsupervised representation learning (URL), particularly for *irregularly sampled* time series with *variable lengths*.

Our work explicitly focuses on learning shape-based representation of time series. First, we propose to view the shape of a time series not merely as its curve $\{s_t : \ t \in \mathsf{I}\}$, but as its graph $\mathsf{G}(s) = \{(t, s(t)) : \ t \in \mathsf{I}\}$. Then, building on the shape analysis literature [5, 57], we adopt the Large Deformation Diffeomorphic Metric Mapping (LDDMM) framework [5, 57] to analyze these graphs. The core idea is to represent each element $\mathsf{G}(s^j)$ of a dataset $(s^j)_{j\in[N]}$ as the transformation of a reference graph $\mathsf{G}(\mathbf{s}_0)$ by a diffeomorphism $\phi_j$, i.e. $\mathsf{G}(s^j) \sim \phi_j.\mathsf{G}(\mathbf{s}_0)$. The diffeomorphism $\phi_j$ is learned by integrating an ordinary differential equation parameterized by a Reproducing Kernel Hilbert Space (RKHS). The parameters $(\alpha_j)_{j\in[N]}$ encoding the diffemorphisms $(\phi_j)_{j\in[N]}$ yield the representation features of the graphs $(\mathsf{G}(s^j))_{j\in[N]}$. Finally, these shape-encoding features can be used as inputs to any statistical or machine-learning model.

However, a time series graph transformation by a general diffeomorphism is not always a time series graph, see e.g. Figure 1, thus a time series graph is more than a simple curve [23]. Our contributions arise from this observation: we specify the class of diffeomorphisms to consider and show how to learn them. This change is fruitful in representing transformations of time series graphs as illustrated in Figure 2.

Our contributions can be summarized as follows:

- We propose an unsupervised method (TS-LDDMM) to analyze the inter-individual variability of shapes in a time series dataset (Section 4). In particular, the method can handle multivariate time series *irregularly sampled* and with *variable sizes*.

- We motivate our extension of LDDMM to time series by introducing a theoretical framework with a representation theorem for time series graph (Theorem 1) and kernels related to their structure (Lemma 1).

- We demonstrate the identifiability of the model by estimating the true generating parameter of synthetic data, and we highlight the sensitivity of our method concerning its hyperparameters (Appendix G.1), also providing guidelines for tuning (Appendix D).

- We highlight the *interpretability* of TS-LDDMM for studying the inter-individual variability in a clinical dataset (Section 5).

- We illustrate the quantitative interest of such representation on classification tasks on real shape-based datasets with regular and irregular sampling (Appendices H and I).

## 2   Notations

We denote by integer ranges by $[k : l] = \{k, \ldots, l\} \subset \mathcal{P}(\mathbb{Z})$ and $[l] = [1 : l]$ with $k, l \in \mathbb{N}$, by $\mathsf{C}^m(\mathsf{I}, \mathsf{E})$ the set of $m$-times continously differentiable function defined on an open set $\mathsf{U}$ to a normed vector space $\mathsf{E}$, by $||u||_\infty = \sup_{x\in\mathsf{U}} |u(x)|$ for any bounded function $u : \mathsf{U} \to \mathsf{E}$, and by $\mathbb{N}_{>0}$ is the set of positive integers.

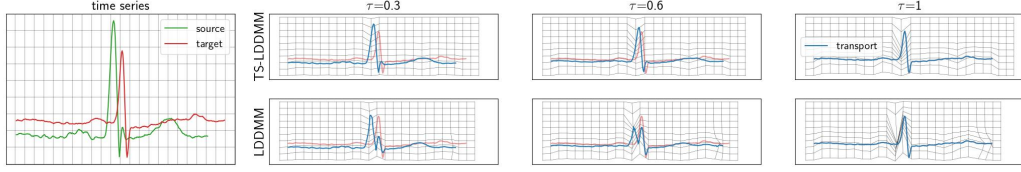

Figure 2: LDDMM and TS-LDDMM are applied to ECG data. We observe that LDDMM, using a general Gaussian kernel, does not learn the time translation of the first spike but changes the space values, i.e., one spike disappears before emerging at a translated position. At the same time, TS-LDDMM handles the time change in the shape. This difference of *deformations* implies differences in features *representations*.

## 3 Background on LDDMM

In this part, we expose how to learn the diffeomorphisms $(\phi_j)_{j \in [N]}$ using LDDMM, initially introduced in [5]. In a nutshell, for any $j \in [N]$, $\phi_j$ corresponds to a differential flow related to a learnable velocity field belonging to a well-chosen Reproducing Kernel Hilbert Space (RKHS).

In the next section, time series are going to be represented by diffeomorphism parameters $(\alpha_j)_{j \in [N]}$. That is why LDDMM is chosen since it offers a parametrization for diffeomorphisms that is sparse and interpretable, two features particularly relevant in the biomedical context.

The basic problem that we consider in this section is the following. Given a set of targets $\mathbf{y} = (y_i)_{i \in [T_2]}$ in $\mathbb{R}^{d'\,2}$, a set of starting points $\mathbf{x} = (x_i)_{i \in [T_1]}$ in $\mathbb{R}^{d'}$, we aim to find a diffeomorphism $\phi$ such that the finite set of points $\mathbf{y}$ is similar in a certain sense to the set of finite sets of transformed points $\phi \cdot \mathbf{x} = (\phi(x_i))_{i \in [T_1]}$. The function $\phi$ is occasionally referred to as a *deformation*. In general, these sets $\mathbf{x}, \mathbf{y}$ are meshes of continuous objects, e.g., surfaces, curves, images, and so on.

**Representing diffeomorpshims as deformations.** Such *deformations* $\phi$ are constructed via differential flow equations, for any $x_0 \in \mathbb{R}^{d'}$ and $\tau \in [0,1]$:

$$\frac{\mathrm{d}X(\tau)}{\mathrm{d}\tau} = v_\tau(X(\tau)), \quad X(0) = x_0 \,, \phi_\tau^v(x_0) = X(\tau), \quad \phi^v = \phi_1^v \,, \tag{1}$$

where the velocity field is $v : \tau \in [0,1] \mapsto v_\tau \in \mathsf{V}$ and $\mathsf{V}$ is a Hilbert space of continuously differentiable function on $\mathbb{R}^{d'}$. If $||\,\mathrm{d}u||_\infty + ||u||_\infty \leq ||u||_\mathsf{V}$ for any $u \in \mathsf{V}$ and $v \in \mathrm{L}^2([0,1],\mathsf{V}) = \{v \in \mathrm{C}^0([0,1],\mathsf{V}) : \int_0^1 ||v_\tau||_\mathsf{V}^2 \,\mathrm{d}\tau < \infty\}$, by [22, Theorem 5] $\phi^v$ exists and belongs to $\mathcal{D}(\mathbb{R}^{d'})$, where we denote by $\mathcal{D}(\mathsf{O})$ the set of diffeomorpshim defined on an open set $\mathsf{O}$ to $\mathsf{O}$. Therefore, for any choice of $v$, $\phi^v$ defines a valid deformation. This offers a general recipe to construct diffeomorphism given a functional space $\mathsf{V}$.

With this in mind, the velocity field $v$ fitting the data can be estimated by minimizing $v \in \mathrm{L}^2([0,1],\mathsf{V}) \mapsto \mathscr{L}(\phi^v.\mathbf{x}, \mathbf{y})$, where $\mathscr{L}$ is an appropriate loss function. However, two computational challenges arise. First, this optimization problem is ill-posed, and a penalty term is needed to obtain a unique solution. In addition, a parametric family $\mathsf{V}_\Theta \subset \mathrm{L}^2([0,1],\mathsf{V})$, parameterized by $\Theta$, is sought to efficiently solve this minimization problem.

**From deformations to geodesics.** It has been proposed in [40] to interpret $\mathsf{V}$ as a tangent space relative to the group of diffeomorphisms $\mathsf{H} = \{\phi^v : v \in \mathrm{L}^2([0,1],\mathsf{V})\}$. Following this geometric point of view, geodesics can be constructed on $\mathsf{H}$ by using the following squared norm

$$\mathscr{R}^2 : g \in \mathsf{H} \mapsto \inf_{v \in \mathrm{L}^2([0,1],\mathsf{V}): \, g = \phi^v} \int_0^1 ||v_\tau||_\mathsf{V}^2 \,\mathrm{d}\tau \tag{2}$$

By deriving differential constraints related to the minimum of (2) and using Cauchy-Lipschitz conditions, geodesics can be defined only by giving the starting point and the initial velocity $v_0 \in \mathsf{V}$ [40], as straight lines in Euclidean space. Denoting by $\tau \mapsto \rho_{v_0}(\tau) \in \mathsf{H}$ the geodesic starting from

the identity with inital velocity $v_0 \in \mathsf{V}$, the exponential map is defined as $\varphi^{\{v_0\}} \triangleq \rho_{v_0}(1)$. Using $\varphi^{\{v_0\}}$ instead of $\phi^v$, the previous matching problem becomes a *geodesic shooting problem*:

$$\inf_{v_0 \in \mathsf{V}} \mathscr{L}(\varphi^{\{v_0\}}.\mathbf{x}, \mathbf{y}). \tag{3}$$

Using $\varphi^{\{v_0\}}$ instead of $\phi^v$ for any $v \in \mathrm{L}^2([0,1], \mathsf{V})$ regularizes the problem and induces a sparse representation for the learning diffeomorphisms. Moreover, by setting $\mathsf{V}$ as an RKHS, the geodesic shooting problem has a unique solution and becomes tractable, as described in the next section.

**Discrete parametrization of diffeomorpshim.** In this part, $\mathsf{V}$ is chosen as an RKHS [6] generated by a smooth kernel $K$ (e.g., Gaussian). We follow [17] and define a discrete parameterization of the velocity fields to perform geodesics shooting (3). The initial velocity field $v_0$ is chosen as a finite linear combination of the RKHS basis vector fields, $\mathbf{n}_0$ control points $\mathsf{X}_0 = (x_{k,0})_{k \in [\mathbf{n}_0]} \in (\mathbb{R}^{d'})^{\mathbf{n}_0}$ and momentum vectors $\alpha_0 = (\alpha_{k,0})_{k \in [\mathbf{n}_0]} \in (\mathbb{R}^{d'})^{\mathbf{n}_0}$ are defined such that for any $x \in \mathbb{R}^{d'}$,

$$v_0(\alpha_0, \mathsf{X}_0)(x) = \sum_{k=1}^{\mathbf{n}_0} K(x, x_{k,0})\alpha_{k,0} . \tag{4}$$

In our applications, the control points $(x_{k,0})_{k \in [\mathbf{n}_0]}$ can be understood as the discretized graph $(t_k, \mathbf{s}_0(t_k))_{k \in [\mathbf{n}_0]}$ of a starting time series $\mathbf{s}_0$. With this parametrization of $v_0$, [40] show that the velocity field $v$ of the solution of (3) keeps the same structure along time, such that for any $x \in \mathbb{R}^{d'}$ and $\tau \in [0,1]$,

$$v_\tau(x) = \sum_{k=1}^{\mathbf{n}_0} K(x, x_k(\tau))\alpha_k(\tau) ,$$

$$\begin{cases} \dfrac{\mathrm{d}x_k(\tau)}{\mathrm{d}\tau} = v_\tau(x_k(\tau)) , \quad \dfrac{\mathrm{d}\alpha_k(\tau)}{\mathrm{d}\tau} = -\sum_{k=1}^{\mathbf{n}_0} \mathrm{d}_{x_k(\tau)}K(x_k(\tau), x_l(\tau))\alpha_l(\tau)^\top \alpha_k(\tau) \\ \alpha_k(0) = \alpha_{k,0}, \quad x_k(0) = x_{k,0} , k \in [\mathbf{n}_0] \end{cases} \tag{5}$$

These equations are derived from the hamiltonian $H : (\alpha_k, x_k)_{k \in [\mathbf{n}_0]} \mapsto \sum_{k,l=1}^{\mathbf{n}_0} \alpha_k^\top K(x_k, x_l)\alpha_l$, such that the velocity norm is preserved $||v_\tau||_\mathsf{V} = ||v_0||_\mathsf{V}$ for any $\tau \in [0,1]$. By (5), the velocity field related to a geodesic $v^*$ is fully parametrized by its initial control points and momentum $(x_{k,0}, \alpha_{k,0})_{k \in [\mathbf{n}_0]}$. Thus, given a set of targets $\mathbf{y} = (y_i)_{i \in [T_2]}$ in $\mathbb{R}^{d'}$, a set of starting points $\mathbf{x} = (x_{i,0})_{i \in [T_1]}$ in $\mathbb{R}^{d'}$, a RKHS's kernel $K : \mathbb{R}^{d'} \times \mathbb{R}^{d'} \to \mathbb{R}^{d' \times d'}$, a distance on sets $\mathscr{L}$, a numerical integration scheme of ODE and a penalty factor $\lambda > 0$, the basic geodesic shooting step minimizes the following function using a gradient descent method:

$$\mathcal{F}_{\mathbf{x},\mathbf{y}} : (\alpha_k)_{k \in [T_1]} \mapsto \mathscr{L}\left(\varphi^{\{v_0\}}.\mathbf{x}, \mathbf{y}\right) + \lambda ||v_0||_\mathsf{V}^2 , \tag{6}$$

where $v_0$ is defined by (4) and $\varphi^{\{v_0\}}.\mathbf{x}$ is the result of the numerical integration of (5) using control points $\mathbf{x}$ and initial momentums $(\alpha_k)_{k \in [T_1]}$.

**Relation to Continuous Normalizing Flows.** One particular popular choice to address the problem of learning a diffeomorphism or a velocity field is Normalizing Flows [47, 32] (NF) or their continuous counterpart [13, 24, 48] (CNF). However, we do not rely on this class of learning algorithms for several reasons. Indeed, existing and simple normalizing flows are not suitable for the type of data that we are interested in this paper [19, 16]. In addition, they are primarily designed to have tractable Jacobian functions, while we do not require such property in our applications. Finally, the use of a differential flow solution of an ODE (1) trick is also at the basis of CNF, which then consists of learning a velocity field to address in fitting the data through a loss aiming to address the problem at hand. Nevertheless, the main difference between CNF and LDDMM lies in the parametrization of the velocity field. LDDMM uses kernels to derive closed form formula and enhance interpretability while NF and CNF take advantage of deep neural networks to scale with large dataset in high dimensions.

## 4 Methodology

We consider in this paper observations which consist in a population of $N$ multivariate time series, for any $j \in [N]$, $s^j \in \mathrm{C}^1(\mathsf{I}_j, \mathbb{R}^d)$. However, we can only access a $n_j$-samples $\tilde{s}^j = (\tilde{s}_i^j = s^j(t_i^j))_{i \in [n_j]}$

collected at timestamps $(t_i^j)_{i \in [n_j]}$ for any $j \in [N]$. Note that **the number of samples $n_j$ is not necessarily the same across individuals** and the timestamps can be **irregularly sampled**. We assume the time series population is globally homogeneous regarding their "shapes" even if inter-individual variability exists. Intuitively speaking, the "shape" of a time series $s : I \to \mathbb{R}^d$ is encoded in its graphs $G(s)$ defined as the set $\{(t, s(t)) : t \in I\}$ and not only in its values $s(I) = \{s(t) : t \in I\}$ since the time axis is crucial. As a motivating use-case, $s^j$ can be the time series of a heartbeat extracted from an individual's electrocardiogram (ECG), see Figure 2. The homogeneity in a resulting dataset comes from the fact that humans have similar shapes of heartbeat [61, 37].

**The deformation problem.**    In this paper, we aim to study the inter-individual variability in the dataset by finding a relevant representation of each time series. Inspired from the framework of shape analysis [57], addressing similar problems in morphology, we suggest to represent each time series' graph $G(s^j)$ as the transformation of a reference graph $G(\mathbf{s}_0)$, related to a time series $\mathbf{s}_0 : I \to \mathbb{R}^d$, by a diffeomorphism $\phi_j$ on $\mathbb{R}^{d+1}$, for any $j \in [N]$,

$$\phi_j . G(\mathbf{s}_0) = \{\phi_j (t, \mathbf{s}_0(t)), \ t \in I\} . \tag{7}$$

$\mathbf{s}_0$ will be understood as the typical representative shape common to the collection of time series $(s^j)_{j \in [N]}$. As $\mathbf{s}_0$ is supposed to be fixed, then the representation of the time series $(s^j)_{j \in [N]}$ boils down to the one of the transformation $(\phi_j)_{j \in [N]}$. We aim to learn $G(\mathbf{s}_0)$ and $(\phi_j)_{j \in [N]}$.

**Optimization related to** (7).    Defining the *discretized graphs* of the time series $(s^j)_{j \in [N]}$ and a discretization of the reference graph $G(\mathbf{s}_0)$ as, for any $j \in [N]$,

$$\mathbf{y}_j = G(\tilde{s}^j) = (t_i^j, \tilde{s}_i^j)_{i \in [n_j]} \in (\mathbb{R}^{d+1})^{n_j}, \quad \tilde{G}_0 = (t_i^0, \tilde{s}_i^0)_{i \in [\mathbf{n}_0]} \in (\mathbb{R}^{d+1})^{\mathbf{n}_0} ,$$

with $\mathbf{n}_0 = \mathrm{median}((n_j)_{j \in [N]})$, the representation problem given in (7) boils down solving:

$$\mathrm{argmin}_{\tilde{G}_0, (\alpha_k^j)_{k \in [\mathbf{n}_0]}^{j \in [N]}} \sum_{j=1}^{N} \mathcal{F}_{\tilde{G}_0, \mathbf{y}_j} \left( (\alpha_k^j)_{k \in [\mathbf{n}_0]} \right) , \tag{8}$$

which is carried out by gradient descent on the control points $\tilde{G}_0$ and the momentums $\alpha_j = (\alpha_k^j)_{k \in [\mathbf{n}_0]}$ for any $j \in [N]$, initialized by a dataset's time series graph of size $\mathbf{n}_0$ and by $0_{(d+1)\mathbf{n}_0}$ respectively. The optimization hyperparameter details are given in Appendix E.1. The result of the minimization $\tilde{G}_0$ is then considered as the $\mathbf{n}_0$-samples of a common time series $\mathbf{s}_0$ and the momentums $\alpha_j$ encoding $\phi_j$ yields a feature vector in $\mathbb{R}^{d\mathbf{n}_0}$ of $s^j$ for any $j \in [N]$. Finally, the vectors $(\alpha_j)_{j \in [N]}$ can be analyzed with any statistical or machine learning tools such as Principal Components Analysis (PCA), Latent Discriminant Analysis (LDA), longitudinal data analysis and so on.

Nevertheless, (8) asks to define a kernel and a loss in order to perform geodesic shooting (6), which is the purpose of the following subsection.

## 4.1    Application of LDDMM to time series analysis: TS-LDDMM

This section presents our theoretical contribution: we tailor the LDDMM framework to handle time series data. The reason is that applying a general diffeomorphism $\phi$ from $\mathbb{R}^{d+1}$ to a time series' graph $G(s)$ can result in a set $\phi . G(s)$ that does not correspond to the graph of any time series, as illustrated in the Figure 1. Thus, time series graphs have more structure than a simple 1D curve [23] and deserve their unique analysis, which will prove fruitful as demonstrated in Section 5.

To address this challenge, we need to identify an RKHS kernel $K : \mathbb{R}^{d+1} \times \mathbb{R}^{d+1} \to \mathbb{R}^{(d+1)^2}$ that generates deformations preserving the structure of the time series graph. This goal motivates us to clarify, in Theorem 1, the specific representation of diffeomorphisms we require before presenting a class of kernels that produce deformations with this representation.

Similarly, selecting a loss function on sets $\mathscr{L}$ that considers the temporal evolution in a time series' graph is crucial for meaningful comparisons with time series data. Consequently, we introduce the oriented Varifold distance.

**A representation separating space and time.**    We prove that two time series graphs can always be linked by a time transformation composed with a space transformation. Moreover, a time series graph transformed by this kind of transformation is always a time series graph. We define $\Psi_\gamma \in$

$\mathcal{D}(\mathbb{R}^{d+1}) : (t,x) \in \mathbb{R}^{d+1} \to (\gamma(t),x)$ for any $\gamma \in \mathcal{D}(\mathbb{R})$ and $\Phi_f : (t,x) \in \mathbb{R}^{d+1} \to (t,f(t,x))$ for any $f \in C^1(\mathbb{R}^{d+1},\mathbb{R}^d)$. We have the following representation theorem. All proofs are given in Appendix B.

Denote by $\mathsf{G}(s) \triangleq \{(t,s(t)) : t \in \mathsf{I}\}$ the graph of a time series $s : \mathsf{I} \to \mathbb{R}^d$ and $\phi.\mathsf{G}(s) \triangleq \{\phi(t,s(t)) : t \in \mathsf{I}\}$ the action of $\phi \in \mathcal{D}(\mathbb{R}^{d+1})$ on $\mathsf{G}(s)$.

**Theorem 1.** *Let $s : \mathsf{J} \to \mathbb{R}^d$ and $\mathbf{s}_0 : \mathsf{I} \to \mathbb{R}^d$ be two continuously differentiable time seriess with $\mathsf{I}, \mathsf{J}$ two intervals of $\mathbb{R}$. There exist $f \in C^1(\mathbb{R}^{d+1},\mathbb{R}^d)$ and $\gamma \in \mathcal{D}(\mathbb{R})$ such that $\gamma(\mathsf{I}) = \mathsf{J}$ and $\Phi_f \in \mathcal{D}(\mathbb{R}^{d+1})$,*

$$\mathsf{G}(s) = \Pi_{\gamma,f}.\mathsf{G}(\mathbf{s}_0), \ \Pi_{\gamma,f} = \Psi_\gamma \circ \Phi_f.$$

*Moreover, for any $\bar{f} \in C^1(\mathbb{R}^{d+1},\mathbb{R}^d)$ and $\bar{\gamma} \in \mathcal{D}(\mathbb{R})$, there exists a continuously differentiable time series $\bar{s}$ such that $\mathsf{G}(\bar{s}) = \Pi_{\bar{\gamma},\bar{f}}.\mathsf{G}(\mathbf{s}_0)$*

**Remark 2.** *Note that for any $\gamma \in \mathcal{D}(\mathbb{R})$ and $s \in C^0(\mathsf{I},\mathbb{R}^d)$,*

$$\{(\gamma(t),s(t)), \ t \in \mathsf{I}\} = \{(t,s \circ \gamma^{-1}(t)) : \ t \in \gamma(\mathsf{I})\} \ .$$

*As a result, $\Psi_\gamma$ can be understood as a temporal reparametrization and $\Phi_f$ encodes the transformation about the space.*

**Choice for the kernel associated with the RKHS $\mathsf{V}$**  As depicted on Figure 1-2, we can not use any kernel $K$ to apply the previous methodology to learn deformations on time series' graphs. We describe and motivate our choice in this paragraph. Denote the one-dimensional Gaussian kernel by $K_\sigma^{(a)}(x,y) = \exp(-|x-y|^2/\sigma)$ for any $(x,y) \in (\mathbb{R}^a)^2$, $a \in \mathbb{N}$ and $\sigma > 0$. To solve the geodesic shooting problem (6) on $\mathbb{R}^{d+1}$, we consider for $\mathsf{V}$ the RKHS associated with the kernel defined for any $(t,x),(t',x') \in (\mathbb{R}^{d+1})^2$:

$$K_{\mathsf{G}}((t,x),(t',x')) = \begin{pmatrix} c_0 K_{\text{time}} & 0 \\ 0 & c_1 K_{\text{space}} \end{pmatrix} , \tag{9}$$

$$K_{\text{space}} = K_{\sigma_{T,1}}^{(1)}(t,t')K_{\sigma_x}^{(d)}(x,x')\mathrm{I}_d , K_{\text{time}} = K_{\sigma_{T,0}}^{(1)}(t,t') ,$$

parametrized by the widths $\sigma_{T,0}, \sigma_{T,1}, \sigma_x > 0$ and the constants $c_0, c_1 > 0$. This choice for $K_{\mathsf{G}}$ is motivated by the representation Theorem 1 and the following result.

**Lemma 1.** *If we denote by $\mathsf{V}$ the RKHS associated with the kernel $K_{\mathsf{G}}$, then for any vector field $v$ generated by (5) with $v_0$ satisfying (4), there exist $\gamma \in \mathsf{D}(\mathbb{R})$ and $f \in C^1(\mathbb{R}^{d+1},\mathbb{R}^d)$ such that $\phi^v = \Psi_\gamma \circ \Phi_f$.*

Instead of Gaussian kernels, other types of smooth kernels can be selected as long as the structure (9) is respected.

**Remark 3.** *With this choice of kernel, the features associated with the time transformation can be extracted from the momentums $(\alpha_{k,0})_{k \in [\mathbf{n}_0]} \in (\mathbb{R}^{d+1})^{\mathbf{n}_0}$ in (4) by taking the coordinates related to time. However, the features related to the space transformation are not only in the space coordinates since the related kernel $K_{space}$ depends on time as well. The kernel's representation has been carefully designed to integrate both space and time, while ensuring that time remains independent of space. Initially, we considered separating the spatial and temporal components. However, post-hoc analysis of such a representation proved to be challenging. The separated spatial and temporal representations are correlated, and understanding this correlation is essential for interpreting the data. As a result, concatenating the two representations becomes necessary, though there is no straightforward method for doing so, as they are not commensurable. Consequently, we opted for a representation that inherently integrates both space and time.*

In Appendix D, we give guidelines for selecting the hyperparameters $(\sigma_{T,0}, \sigma_{T,1}, \sigma_x, c_0, c_1)$.

**Loss**  This section specifies the distance function $\mathscr{L}$ introduced in the loss function defined in (6).

In practice, we can only access discretized graphs of time series, $(t_i^j, \tilde{s}_i^j)_{i \in [n_j]}$ for any $j \in [N]$, that are potentially of different sizes $n_j$ and sampled at different timestamps $(t_i^j)_{i \in [n_j]}$ for any $j \in [N]$. Usual metrics, such as the Euclidean distance, are not appealing as they make the underlying assumptions of equal size sets and the existence of a pairing between points. Distances between measures on sets

(taking the empirical distribution), such as Maximum Mean Discaprency (MMD) [18, 9], alleviate those issues; however, MMD only accounts for positional information and lacks information about the time evolution between sampled points. A classical data fidelity metric from shape analysis corresponding to the distance between *oriented varifolds* associated with curves alleviates this last issue [30]. Intuitively, an oriented varifold is a measure that accounts for positional and tangential information about the underlying curves at sample points. More details and information about *oriented varifolds* can be found in Appendix C.

More precisely, given two sets $\mathsf{G}_0 = (g_i^0)_{i\in[T_0]}, \mathsf{G}_1 = (g_i^1)_{i\in[T_1]} \in (\mathbb{R}^{d+1})^{T_1}$ and a kernel[3] $k : (\mathbb{R}^{d+1} \times \mathbb{S}^d)^2 \to \mathbb{R}$ verifying [30, Proposition 2 & 4], for any $\xi \in \{0,1\}$ and $i \in [T_\xi - 1]$, denoting the center and length of the $i^{th}$ segment $[g_i^\xi, g_{i+1}^\xi]$ by $c_i^\xi = (g_i^\xi + g_{i+1}^\xi)/2$, $l_i^\xi = \|g_{i+1}^\xi - g_i^\xi\|$, and $\overrightarrow{v_i}^\xi = (g_{i+1}^\xi - g_i^\xi)/l_i^\xi$, the varifold distance between $\mathsf{G}_0$ and $\mathsf{G}_1$ is defined as,

$$d_{\mathsf{W}^*}^2(\mathsf{G}_0, \mathsf{G}_1) = \sum_{i,j=1}^{T_0-1} l_i^0 k((c_i^0, \overrightarrow{v_i}^0), (c_j^0, \overrightarrow{v_j}^0))l_j^0 - 2\sum_{i=1}^{T_0-1}\sum_{j=1}^{T_1-1} l_i^0 k((c_i^0, \overrightarrow{v_i}^0), (c_j^1, \overrightarrow{v_j}^1))l_j^1$$
$$+ \sum_{i,j=1}^{T_1-1} l_i^1 k((c_i^1, \overrightarrow{v_i}^1), (c_j^1, \overrightarrow{v_j}^1))l_j^1$$

In practice, we set the kernel $k$ as the product of two anisotropic Gaussian kernels, $k_{\text{pos}}$ and $k_{\text{dir}}$, such that for any $(x, \overrightarrow{u}), (y, \overrightarrow{v}) \in (\mathbb{R}^{d+1} \times \mathbb{S}^d)^2$

$$k((x, \overrightarrow{u}), (y, \overrightarrow{v})) = k_{\text{pos}}(x, y)k_{\text{dir}}(\overrightarrow{u}, \overrightarrow{v}) .$$

Note that the loss kernel $k$ has nothing to do with the velocity field kernel denoted by $K_\mathsf{G}$ or $K$ specified in Section 4.1. Finally, we define the data fidelity loss function, $\mathscr{L}$, as a sum of $d_{\mathsf{W}^*}^2$ using different kernel's width parameters $\sigma$ to incorporate multiscale information. $\mathscr{L}$ is indeed differentiable with respect to its first variable. The specific kernels $k_{\text{pos}}, k_{\text{dir}}$ that we use in our experiments are given Appendix C.1. For further readings on curves and surface representation as varifolds, readers can refer to [30, 12].

A pedagogical online application is available to inspect the effect of hyperprameters on geodesic shooting (5) and registration (6).

## 5 Experiments

The source code is available on Github[4]. For conciseness, several experiments are relegated in appendix:

1. **TS-LDDMM representation identifiability, Appendix G:** On synthetic data, we evaluate the ability of our method to retrieve the parameter $v_0^*$ that encodes the deformation $\varphi^{\{v_0^*\}}$ acting on a time series graph $\mathsf{G}$ by solving the geodesic shooting problem (6) between $\mathsf{G}$ and $\varphi^{\{v_0^*\}}.\mathsf{G}$. **Results** show that TS-LDDMM representations are identifiable or weakly identifiable depending on the velocity field kernel $K_G$ specification.

2. **Robustness to irregular sampling, Appendix H:** We compare the robustness of TS-LDDMM representation with 9 URL methods handling irregularly sampled multivariate time series on 15 shape-based datasets (7 univariates & 8 multivariates). We assess methods' classification performances under regular sampling (0% missing rate) and three irregular sampling regimes (30%, 50%, and 70% missing rates), according to the protocol depicted in [31]. **Results** show that our method, TS-LDDMM, outperforms all methods for sampling regimes with missing rates: 0%, 30%, and 50%.

3. **Classification benchmark on regularly sampled datasets, Appendix I:** We compare performances of a kernel support vector machine (SVC) algorithm based on TS-LDDMM representation with 3 state-of-the-art classification methods from shape analysis on 15 shape-based datasets (7 univariates & 8 multivariates). **Results** show that the TS-LDDMM-based method outperforms other methods (best performances over 13 datasets), making TS-LDDMM representation relevant for time series shape analysis.

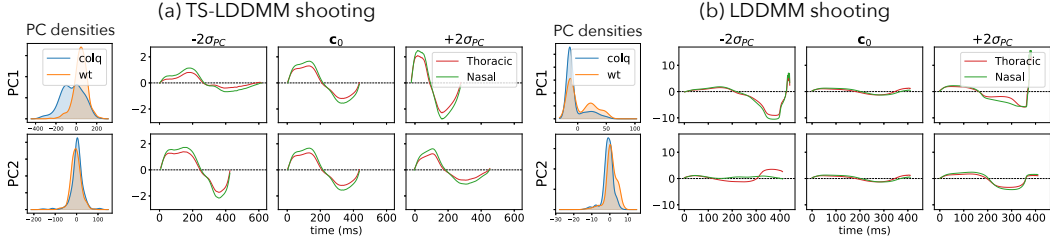

Figure 3: Analysis of the two principal components (PC) related to mice ventilation before exposure with TS-LDDMM representations **(a)**, and LDDMM **(b)**. In both cases and for all PC, the left plot displays PC densities according to mice genotype and right plot displays deformations of the reference graph $\mathbf{c}_0$ along each PC.

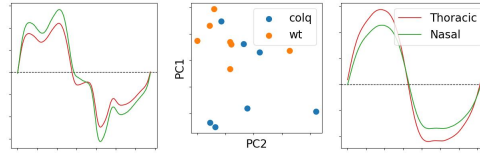

(a) ColQ cycle   (b) PC1 vs PC2   (c) WT cycle

Figure 4: **(a)** a ColQ respiratory cycle sample. **(b)** Referent respiratory cycle of individual mouse $\mathbf{c}_0^j$ in the TS-LDDMM PC1-PC2 coordinates system of $\mathbf{c}_0$. **(c)** a WT respiratory cycle sample.

4. **Noise sensitivity for learning the reference graph, Appendix J:** We evaluate the noise sensitivity of TS-LDDMM and Shape-FPCA [60] for learning the reference graph on a synthetic dataset and for several levels of additive Gaussian noise. **Results** show that both methods are sensitive to noise. However, TS-LDDMM preserves the overall shape while shape-FPCA alters the shape depending on the noise level.

## 5.1 Interpretability: mice ventilation analysis

This experiment highlights the *interpretability* of TS-LDDMM representation for studying the inter-individual variability in biomedical applications. We consider a time series dataset monitoring the evolution of mice's nasal and thoracic airflow when exposed to a drug altering respiration [41]. The dataset includes recordings of 7 control mice (WT) and 7 mutant mice (ColQ) with an enzyme deficiency. The enzyme is involved in the respiration regulation, and the drug inhibits its activity. For each mouse, airflows were monitored for 15 to 20 minutes before the drug exposure and then for 35 to 40 minutes. A complete description of the dataset is given in the Appendix F.1.

**Experimental protocol.** We considered two experimental scenarios; the first focuses on mice ventilation before exposure to explore the inter-individual and genotype-specific variabilities. The second focuses on whole recordings to analyze the evolution of mice's ventilation after drug exposure. In both cases, the baseline protocol consists of first extracting $N$ respiratory cycles from the datasets with the procedure described in [21]. Then, learning the referent respiratory cycle $\mathbf{c}_0$ and the representations of respiratory cycles $(\boldsymbol{\alpha}_0^j)_{j \in [N]}$ by solving (8) using TS-LDDMM. $\boldsymbol{\alpha}_0^j$ being the momentum of the initial velocity field of the geodesic encodings the diffeomorphisms mapping $\mathbf{c}_0$ to the $j^{th}$ respiratory cycle. Finally, performing a Kernel-PCA on the initial velocity fields (4) belonging to $\mathsf{V}$ and encoded by the pairs $(\boldsymbol{\alpha}_0^j, \mathbf{c}_0)_{j \in [N]}$. The first experiment includes $N_1 = 700$ cycles collected before exposure. The second experiment includes $N_2 = 1400$ cycles with 25% (resp. 75%) before (resp. after) exposure. We also performed the first experimental scenario with LDDMM representation, and Appendix K describes the settings of both methods. Essentially, varifold losses are identical for both methods, and the velocity field kernels are set to encompass time and space scales. in addition, In addition, Appendix K presents a comparison between TS-LDDMM and Shape-FPCA on the second scenario.

**Geodesic shooting along principal component directions.** Any principal component (PC), noted $v_0^{pc}$, from a kernel-PCA in $\mathsf{V}$, is itself an initial velocity field encoded by a pair $(\mathbf{c}_0, \boldsymbol{\alpha}_0^{pc})$. PCs encode the principal axis of deformations, and it is possible to shoot along the geodesic they encode with the differential equations (5), enabling interpretation of the main sources of deformations.

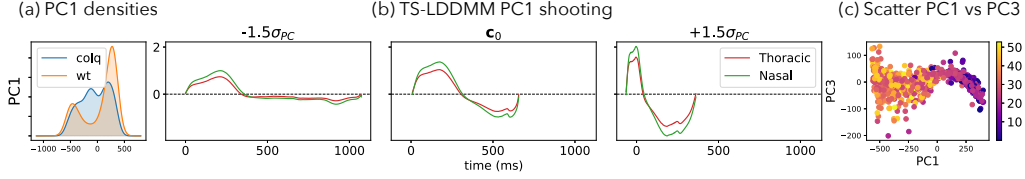

Figure 5: Analysis of the first Principal Component (PC1) related to mice ventilation before and after exposure with TS-LDDMM representations. **(a)** displays PC densities per mice genotype, **(b)** illustrates deformations of the reference respiratory cycle $c_0$ along PC1, and **(c)** displays all respiratory cycles with respect to time in PC1 and PC3 coordinates

**Mice ventilation before exposure.** We focus on the analysis of the two first Principal Components (PC) for TS-LDDMM (Figure 3a) and LDDMM (Figure 3b). Looking at the geodesic shooting along PCs, Figure 3 shows that principal components learned with TS-LDMM lead to deformations that remain respiratory cycles. In contrast, deformations learned with LDDMM are challenging to interpret as respiratory cycles. The LDDMM velocity field kernel is a Gaussian anisotropic kernel that accounts for time and space scales; however, the entanglement of time and space dimensions in the kernel does not guarantee the graph structure, and it makes the convergence of the method complex (relative varifold loss error: TS-LDDMM: 0.06, LDDMM: 0.11).

Regarding TS-LDDMM Figure 3a, its PCs refer to deformations directions carrying different physiological meanings. Indeed, the geodesic shooting along these directions indicates that PC1 accounts for variations of the total duration of a respiratory cycle, while PC2 expresses the trade-off between inspiration and expiration duration. In addition, the distribution of ColQ respiratory cycles along PC1 is wider than in WT mice, indicating that the adaptation of mutant mice to their enzyme deficiency is variable. This observation can also be seen in Figure 4b where a referent respiratory cycle $c_0^j$ is learned by solving (8) for each mouse and is encoded in the (PC1,PC2) coordinate system of $c_0$ by registration (3). Indeed, the average respiratory cycles of ColQ mice are more spread out than WT mice's. Going back to the densities of PC1, ColQ mice distribution has a heavier tail toward negative values compared to WT mice. When shooting in the opposite direction of PC1, we can observe that the inspiration is divided into two steps. Congruently with [21], such inspirations indicate motor control difficulties due to enzyme deficiency. Figure 4a is an example of ColQ respiratory cycle with negative PC1 coordinate.

**Mice ventilation evolution after drug exposure.** This experiment focuses on the first principal components learned from TS-LDDDM representations of respiratory cycles randomly sampled before and after drug exposure. Figure 5a illustrates the geodesic shootings along PC1. Again, PC1 accounts for variations in respiratory cycle duration, but more importantly, it can be observed on the deformation at -1.5 $\sigma_{PC}$ the apparition of a long pause after inspiration. Congruently, Figure 5c indicates that pauses appear after drug exposure as cycles with negative PC1 values mainly occur after 20 minutes and present more variability along PC3. In addition, Figure 5b shows a bimodal distribution for WT mice with one of the peaks in the negative values. This peak was not observed in the previous experiment Figure 3a. It indicates that pauses after inspiration are prevalent in WT mice after drug exposure. On the other hand, the distributions of ColQ mice's respiratory cycles along PC1 in both experiments are similar and account for the same deformation, suggesting that ColQ mice weakly react to the drug exposure as they already adapt their enzyme deficiency.

**Experiment Conclusion.** Analyzing mice ventilation with TS-LDDMM representation highlights the method's ability to create meaningful interaction between experts and the data. Indeed, combining statistical and visual results shows that main deformations carry physiological meaning, enabling the characterization of some mice genotypes and the effects of drug exposure.

# 6 Related Works

Shape analysis focuses on statistical analysis of mathematical objects invariant under some deformations like rotations, dilations, or time parameterization. The main idea is to represent these objects in a complete Riemannian manifold $(\mathcal{M}, \mathbf{g})$ with a metric $\mathbf{g}$ adapted to the geometry of the problem [40]. Then, any set of points in $\mathcal{M}$ can be represented as points in the tangent space of their Frechet mean $\mathbf{m}_0$ [44, 33] by considering their logarithms. The goal is to find a well-suited Riemannian structure according to the nature of the studied object.

LDDMM framework is a relevant shape analysis tool to represent curves as depicted in [23]. However, graphs of time series are a well-structured type of curve due to the inclusion of the temporal dimension that requires specific care (Figure 1). In a similar vein, Qiu *et al* [46] proposes a method for tracking anatomical shape changes in serial images using LDDMM. They include temporal evolution, but not for the same purpose: the aim is to perform longitudinal modeling of brain images.

Leaving the LDDMM representation, the results of [53, 26] address the representation of curves with the Square-Root Velocity (SRV) representation. However, the SRV representation is applied after reparametrization of the temporal dimension of the unit length segment. Consequently, the graph structure of the time series is not respected, and the original time evolution of the time series is not encoded in the final representation. Very recently, in a functional data analysis (FDA) framework, a paper [60] (Shape-FPCA) improved by representing the original time evolution. However, the space and time representations remain correlated, complicating post-hoc analysis, as discussed in Remark 3. Additionally, this method is tailored for *continuous objects* and applies only to time series of the *same length*, making the estimation more sensitive to noise. This issue can be addressed through interpolation, but this approach is not always reliable in sparse and irregular sampling scenarios. Most FDA approaches, as seen in [50, 63, 59], address this challenge using interpolation or basis function expansion. In summary, FDA methods typically separate space and time representations for continuous objects, whereas TS-LDDMM algorithm maintain a discrete-to-discrete analysis, inherently integrating both space and time representations.

Balancing between discrete and continuous elements is a challenging task. In the deep learning literature [13, 31, 56, 29, 36, 1], Neural Ordinary Differential Equations (Neural ODEs) [13] learn continuous latent representations using a vector field parameterized by a neural network, serving as a continuous analog to Residual Networks [64]. This approach was further enhanced by Neural Controlled Differential Equations (Neural CDEs) [31] for handling irregular time series, functioning as continuous-time analogs of RNNs [49]. Extending Neural ODEs, Neural Stochastic Differential Equations (Neural SDEs) introduce regularization effects [36], although optimization remains challenging. Leveraging techniques from continuous-discrete filtering theory, Ansari et al. [1] applied successfully Neural SDEs to irregular time series. Oh *et al.* [43] improved these results by incorporating the concept of controlled paths into the drift term, similar to how Neural CDEs outperform Neural ODEs. With TS-LDDMM, the representation is also derived from an ODE, but the velocity field is parameterized with kernels and optimized to have a minimal norm, which enhances interpretability.

All these state-of-the-art methods previously mentionned [23, 43, 60, 26] are compared to TS-LDDMM in Appendix H and Appendix I.

Compared to the Metamorphosis framework [7], LDDMM framework has weaker assumptions. The 3DMM framework requires that each mesh be re-parametrized into a consistent form where the number of vertices, triangulation, and the anatomical meaning of each vertex are consistent across all meshes, as stated in the introduction of [8]. In our context, we do not need such pre-processing; the time series graph can have different sizes.

# 7    Limitations and conclusion

This paper proposes a feature representation method, TS-LDDMM, designed for shape comparison on homogeneous time series datasets. We show on a real dataset its ability to study, with high interpretability, the inter-individual shape variability. As an unsupervised approach, it is user-friendly and enables knowledge transfer for different supervised tasks such as classification.

Although TS-LDDMM is already competitive for classification, its performances can be leveraged on more heterogeneous datasets using a hierarchical clustering extension, which is relegated for future work.

TS-LDDMM employs kernel computations, which require specific libraries (e.g., KeOps [11]) to be efficient and scalable. However, in our experiments, the time complexity of TS-LDDMM is comparable to that of competitors. It is clear that TS-LDDMM needs to be extended to handle very large datasets with high-dimensional time series (such as videos).

Additionally, TS-LDDMM requires tuning several hyperparameters, though this is a common requirement among competitors [23, 43, 60, 26]. In future work, adaptive methods are expected to be developed to provide a more user-friendly interface.

## Acknowledgments and Disclosure of Funding

This work was supported by grants from Région Ile-de-France (DIM MathInnov). Charles Truong is funded by the PhLAMES chair of ENS Paris-Saclay.

## Footnotes

[1]Complete affiliation: Université Paris Saclay, Université Paris Cité, ENS Paris Saclay, CNRS, SSA, INSERM, Centre Borelli, F-91190, Gif-sur-Yvette, France.

*Corresponding author. Contact at `thibaut.germain@ens-paris-saclay.fr`

[2]Note that we denote by $d' \in \mathbb{N}$ the ambient space

[3]$\mathbb{S}^d = \{x \in \mathbb{R}^{d+1} : |x| = 1\}$

[4]`https://github.com/thibaut-germain/TSLDDMM`

[5]`https://github.com/thibaut-germain/TSLDDMM`

[6]https://github.com/google/jax

[7]https://optax.readthedocs.io/en/latest/

[8]https://timeseriesclassification.com

[9]https://www.aeon-toolkit.org/en/stable/

[10]`https://github.com/yongkyung-oh/Stable-Neural-SDEs`

[11]`https://github.com/ray-project/ray`

[12]`https://scikit-learn.org/stable/`

[13]`https://scikit-learn.org/stable/modules/generated/sklearn.metrics.f1_score.html`

[14]https://fdasrsf-python.readthedocs.io/en/latest/

## References

[1] Abdul Fatir Ansari, Alvin Heng, Andre Lim, and Harold Soh. Neural continuous-discrete state space models for irregularly-sampled time series. In *International Conference on Machine Learning*, pages 926–951. PMLR, 2023.

[2] Asal Asgari. Clustering of clinical multivariate time-series utilizing recent advances in machine-learning. 2023.

[3] Anthony Bagnall, Hoang Anh Dau, Jason Lines, Michael Flynn, James Large, Aaron Bostrom, Paul Southam, and Eamonn Keogh. The uea multivariate time series classification archive, 2018. *arXiv preprint arXiv:1811.00075*, 2018.

[4] Ziv Bar-Joseph, Anthony Gitter, and Itamar Simon. Studying and modelling dynamic biological processes using time-series gene expression data. *Nature Reviews Genetics*, 13(8):552–564, 2012.

[5] M Faisal Beg, Michael I Miller, Alain Trouvé, and Laurent Younes. Computing large deformation metric mappings via geodesic flows of diffeomorphisms. *International journal of computer vision*, 61:139–157, 2005.

[6] Alain Berlinet and Christine Thomas-Agnan. *Reproducing kernel Hilbert spaces in probability and statistics*. Springer Science & Business Media, 2011.

[7] Volker Blanz and Thomas Vetter. Face recognition based on fitting a 3d morphable model. *IEEE Transactions on pattern analysis and machine intelligence*, 25(9):1063–1074, 2003.

[8] James Booth, Anastasios Roussos, Stefanos Zafeiriou, Allan Ponniah, and David Dunaway. A 3d morphable model learnt from 10,000 faces. In *Proceedings of the IEEE conference on computer vision and pattern recognition*, pages 5543–5552, 2016.

[9] Karsten M Borgwardt, Arthur Gretton, Malte J Rasch, Hans-Peter Kriegel, Bernhard Schölkopf, and Alex J Smola. Integrating structured biological data by kernel maximum mean discrepancy. *Bioinformatics*, 22(14):e49–e57, 2006.

[10] Claudio Carmeli, Ernesto De Vito, Alessandro Toigo, and Veronica Umanitá. Vector valued reproducing kernel hilbert spaces and universality. *Analysis and Applications*, 8(01):19–61, 2010.

[11] Benjamin Charlier, Jean Feydy, Joan Alexis Glaunes, François-David Collin, and Ghislain Durif. Kernel operations on the gpu, with autodiff, without memory overflows. *Journal of Machine Learning Research*, 22(74):1–6, 2021.

[12] Nicolas Charon and Alain Trouvé. The varifold representation of nonoriented shapes for diffeomorphic registration. *SIAM journal on Imaging Sciences*, 6(4):2547–2580, 2013.

[13] Ricky TQ Chen, Yulia Rubanova, Jesse Bettencourt, and David K Duvenaud. Neural ordinary differential equations. *Advances in neural information processing systems*, 31, 2018.

[14] Junyoung Chung, Caglar Gulcehre, KyungHyun Cho, and Yoshua Bengio. Empirical evaluation of gated recurrent neural networks on sequence modeling. *arXiv preprint arXiv:1412.3555*, 2014.

[15] Hoang Anh Dau, Anthony Bagnall, Kaveh Kamgar, Chin-Chia Michael Yeh, Yan Zhu, Shaghayegh Gharghabi, Chotirat Ann Ratanamahatana, and Eamonn Keogh. The ucr time series archive. *IEEE/CAA Journal of Automatica Sinica*, 6(6):1293–1305, 2019.

[16] Ruizhi Deng, Bo Chang, Marcus A Brubaker, Greg Mori, and Andreas Lehrmann. Modeling continuous stochastic processes with dynamic normalizing flows. *Advances in Neural Information Processing Systems*, 33:7805–7815, 2020.

[17] Stanley Durrleman, Stéphanie Allassonnière, and Sarang Joshi. Sparse adaptive parameterization of variability in image ensembles. *International Journal of Computer Vision*, 101:161–183, 2013.

[18] Gintare Karolina Dziugaite, Daniel M Roy, and Zoubin Ghahramani. Training generative neural networks via maximum mean discrepancy optimization. *arXiv preprint arXiv:1505.03906*, 2015.

[19] Shibo Feng, Chunyan Miao, Ke Xu, Jiaxiang Wu, Pengcheng Wu, Yang Zhang, and Peilin Zhao. Multi-scale attention flow for probabilistic time series forecasting. *IEEE Transactions on Knowledge and Data Engineering*, 2023.

[20] Jean-Yves Franceschi, Aymeric Dieuleveut, and Martin Jaggi. Unsupervised scalable representation learning for multivariate time series. *Advances in neural information processing systems*, 32, 2019.

[21] Thibaut Germain, Charles Truong, Laurent Oudre, and Eric Krejci. Unsupervised classification of plethysmography signals with advanced visual representations. *Frontiers in Physiology*, 14:781, 2023.

[22] Joan Glaunes. Transport par difféomorphismes de points, de mesures et de courants pour la comparaison de formes et l'anatomie numérique. *These de sciences, Université Paris*, 13, 2005.

[23] Joan Glaunes, Anqi Qiu, Michael I Miller, and Laurent Younes. Large deformation diffeomorphic metric curve mapping. *International journal of computer vision*, 80:317–336, 2008.

[24] Will Grathwohl, Ricky TQ Chen, Jesse Bettencourt, and David Duvenaud. Scalable reversible generative models with free-form continuous dynamics. In *International Conference on Learning Representations*, page 7, 2019.

[25] Ella Guscelli, John I Spicer, and Piero Calosi. The importance of inter-individual variation in predicting species' responses to global change drivers. *Ecology and Evolution*, 9(8):4327–4339, 2019.

[26] Tae-Young Heo, Joon Myoung Lee, Myung Hun Woo, Hyeongseok Lee, and Min Ho Cho. Logistic regression models for elastic shape of curves based on tangent representations. *Journal of the Korean Statistical Society*, pages 1–19, 2024.

[27] Sepp Hochreiter and Jürgen Schmidhuber. Long short-term memory. *Neural computation*, 9(8):1735–1780, 1997.

[28] Heinz Gerd Hoymann. Lung function measurements in rodents in safety pharmacology studies. *Frontiers in pharmacology*, 3:156, 2012.

[29] Junteng Jia and Austin R Benson. Neural jump stochastic differential equations. *Advances in Neural Information Processing Systems*, 32, 2019.

[30] Irene Kaltenmark, Benjamin Charlier, and Nicolas Charon. A general framework for curve and surface comparison and registration with oriented varifolds. In *Proceedings of the IEEE Conference on Computer Vision and Pattern Recognition*, pages 3346–3355, 2017.

[31] Patrick Kidger, James Morrill, James Foster, and Terry Lyons. Neural controlled differential equations for irregular time series. *Advances in Neural Information Processing Systems*, 33:6696–6707, 2020.

[32] Ivan Kobyzev, Simon JD Prince, and Marcus A Brubaker. Normalizing flows: An introduction and review of current methods. *IEEE transactions on pattern analysis and machine intelligence*, 43(11):3964–3979, 2020.

[33] Huiling Le. Locating fréchet means with application to shape spaces. *Advances in Applied Probability*, 33(2):324–338, 2001.

[34] Mathias Lechner and Ramin Hasani. Learning long-term dependencies in irregularly-sampled time series. *arXiv preprint arXiv:2006.04418*, 2020.

[35] Yurim Lee, Eunji Jun, Jaehun Choi, and Heung-Il Suk. Multi-view integrative attention-based deep representation learning for irregular clinical time-series data. *IEEE Journal of Biomedical and Health Informatics*, 26(8):4270–4280, 2022.

[36] Xuanqing Liu, Tesi Xiao, Si Si, Qin Cao, Sanjiv Kumar, and Cho-Jui Hsieh. Neural sde: Stabilizing neural ode networks with stochastic noise. *arXiv preprint arXiv:1906.02355*, 2019.

[37] Putri Madona, Rahmat Ilias Basti, and Muhammad Mahrus Zain. Pqrst wave detection on ecg signals. *Gaceta Sanitaria*, 35:S364–S369, 2021.

[38] Larry Medsker and Lakhmi C Jain. *Recurrent neural networks: design and applications*. CRC press, 1999.

[39] Qianwen Meng, Hangwei Qian, Yong Liu, Yonghui Xu, Zhiqi Shen, and Lizhen Cui. Unsupervised representation learning for time series: A review. *arXiv preprint arXiv:2308.01578*, 2023.

[40] Michael I Miller, Alain Trouvé, and Laurent Younes. Geodesic shooting for computational anatomy. *Journal of mathematical imaging and vision*, 24:209–228, 2006.

[41] Aurélie Nervo, André-Guilhem Calas, Florian Nachon, and Eric Krejci. Respiratory failure triggered by cholinesterase inhibitors may involve activation of a reflex sensory pathway by acetylcholine spillover. *Toxicology*, 424:152232, 2019.

[42] Vit Niennattrakul and Chotirat Ann Ratanamahatana. Inaccuracies of shape averaging method using dynamic time warping for time series data. In *Computational Science–ICCS 2007: 7th International Conference, Beijing, China, May 27-30, 2007, Proceedings, Part I 7*, pages 513–520. Springer, 2007.

[43] YongKyung Oh, Dongyoung Lim, and Sungil Kim. Stable neural stochastic differential equations in analyzing irregular time series data. In *The Twelfth International Conference on Learning Representations*, 2024.

[44] Susovan Pal, Roger P Woods, Suchit Panjiyar, Elizabeth Sowell, Katherine L Narr, and Shantanu H Joshi. A riemannian framework for linear and quadratic discriminant analysis on the tangent space of shapes. In *Proceedings of the IEEE Conference on Computer Vision and Pattern Recognition Workshops*, pages 47–55, 2017.

[45] John Paparrizos and Luis Gravano. k-shape: Efficient and accurate clustering of time series. In *Proceedings of the 2015 ACM SIGMOD international conference on management of data*, pages 1855–1870, 2015.

[46] Anqi Qiu, Marilyn Albert, Laurent Younes, and Michael I Miller. Time sequence diffeomorphic metric mapping and parallel transport track time-dependent shape changes. *NeuroImage*, 45(1):S51–S60, 2009.

[47] Danilo Rezende and Shakir Mohamed. Variational inference with normalizing flows. In *International conference on machine learning*, pages 1530–1538. PMLR, 2015.

[48] Hadi Salman, Payman Yadollahpour, Tom Fletcher, and Kayhan Batmanghelich. Deep diffeomorphic normalizing flows. *arXiv preprint arXiv:1810.03256*, 2018.

[49] Mike Schuster and Kuldip K Paliwal. Bidirectional recurrent neural networks. *IEEE transactions on Signal Processing*, 45(11):2673–2681, 1997.

[50] Han Lin Shang. A survey of functional principal component analysis. *AStA Advances in Statistical Analysis*, 98:121–142, 2014.

[51] Gota Shirato, Natalia Andrienko, and Gennady Andrienko. Identifying, exploring, and interpreting time series shapes in multivariate time intervals. *Visual Informatics*, 7(1):77–91, 2023.

[52] Satya Narayan Shukla and Benjamin M Marlin. Multi-time attention networks for irregularly sampled time series. *arXiv preprint arXiv:2101.10318*, 2021.

[53] Anuj Srivastava, Eric Klassen, Shantanu H Joshi, and Ian H Jermyn. Shape analysis of elastic curves in euclidean spaces. *IEEE transactions on pattern analysis and machine intelligence*, 33(7):1415–1428, 2010.

[54] Sana Tonekaboni, Danny Eytan, and Anna Goldenberg. Unsupervised representation learning for time series with temporal neighborhood coding. *arXiv preprint arXiv:2106.00750*, 2021.

[55] Patara Trirat, Yooju Shin, Junhyeok Kang, Youngeun Nam, Jihye Na, Minyoung Bae, Joeun Kim, Byunghyun Kim, and Jae-Gil Lee. Universal time-series representation learning: A survey. *arXiv preprint arXiv:2401.03717*, 2024.

[56] Belinda Tzen and Maxim Raginsky. Neural stochastic differential equations: Deep latent gaussian models in the diffusion limit. *arXiv preprint arXiv:1905.09883*, 2019.

[57] Marc Vaillant, Michael I Miller, Laurent Younes, and Alain Trouvé. Statistics on diffeomorphisms via tangent space representations. *NeuroImage*, 23:S161–S169, 2004.

[58] Kai Wang, Youjin Zhao, Qingyu Xiong, Min Fan, Guotan Sun, Longkun Ma, Tong Liu, et al. Research on healthy anomaly detection model based on deep learning from multiple time-series physiological signals. *Scientific Programming*, 2016, 2016.

[59] John Warmenhoven, Norma Bargary, Dominik Liebl, Andrew Harrison, Mark A Robinson, Edward Gunning, and Giles Hooker. Pca of waveforms and functional pca: A primer for biomechanics. *Journal of Biomechanics*, 116:110106, 2021.

[60] Yuexuan Wu, Chao Huang, and Anuj Srivastava. Shape-based functional data analysis. *TEST*, 33(1):1–47, 2024.

[61] Can Ye, BVK Vijaya Kumar, and Miguel Tavares Coimbra. Heartbeat classification using morphological and dynamic features of ecg signals. *IEEE Transactions on Biomedical Engineering*, 59(10):2930–2941, 2012.

[62] Lexiang Ye and Eamonn Keogh. Time series shapelets: a new primitive for data mining. In *Proceedings of the 15th ACM SIGKDD international conference on Knowledge discovery and data mining*, pages 947–956, 2009.

[63] Qunqun Yu, Xiaosun Lu, and JS Marron. Principal nested spheres for time-warped functional data analysis. *Journal of Computational and Graphical Statistics*, 26(1):144–151, 2017.

[64] Sergey Zagoruyko and Nikos Komodakis. Wide residual networks. *arXiv preprint arXiv:1605.07146*, 2016.

[65] Juntang Zhuang, Tommy Tang, Yifan Ding, Sekhar C Tatikonda, Nicha Dvornek, Xenophon Papademetris, and James Duncan. Adabelief optimizer: Adapting stepsizes by the belief in observed gradients. *Advances in neural information processing systems*, 33:18795–18806, 2020.

## A  Societal impact

We believe that the paper has a positive societal impact for the following reasons:

- TS-LDDMM is an interpretable method for understanding inter-individual variability in biomedical datasets, potentially offering new insights in medicine.
- TS-LDDMM bridges the gap between the shape analysis community and the unsupervised representation learning (URL) community, fostering potential future collaborations between these fields.

However, the computational cost of the method may raise environmental concerns similar to those associated with deep learning [43]. Additionally, while TS-LDDMM has promising biomedical applications, it could also be misused for creating poison.

## B  Proofs

Denote by $\mathsf{G}(s) \triangleq \{(t, s(t)) : t \in \mathsf{I}\}$ the graph of a time series $s : \mathsf{I} \to \mathbb{R}^d$ and $\phi.\mathsf{G}(s) \triangleq \{\phi(t, s(t)) : t \in \mathsf{I}\}$ the action of $\phi \in \mathcal{D}(\mathbb{R}^{d+1})$ on $\mathsf{G}(s)$.

**Theorem 4.** *Let* $s : \mathsf{J} \to \mathbb{R}^d$ *and* $\mathbf{s}_0 : \mathsf{I} \to \mathbb{R}^d$ *be two continuously differentiable time seriess with* $\mathsf{I}, \mathsf{J}$ *two intervals of* $\mathbb{R}$. *There exist* $f \in \mathrm{C}^1(\mathbb{R}^{d+1}, \mathbb{R}^d)$ *and* $\gamma \in \mathcal{D}(\mathbb{R})$ *such that* $\gamma(\mathsf{I}) = \mathsf{J}$ *and* $\Phi_f \in \mathcal{D}(\mathbb{R}^{d+1})$,

$$\mathsf{G}(s) = \Pi_{\gamma, f}.\mathsf{G}(\mathbf{s}_0), \ \Pi_{\gamma, f} = \Psi_\gamma \circ \Phi_f.$$

*Moreover, for any* $\bar{f} \in \mathrm{C}^1(\mathbb{R}^{d+1}, \mathbb{R}^d)$ *and* $\bar{\gamma} \in \mathcal{D}(\mathbb{R})$, *there exists a continuously differentiable time series* $\bar{s}$ *such that* $\mathsf{G}(\bar{s}) = \Pi_{\bar{\gamma}, \bar{f}}.\mathsf{G}(\mathbf{s}_0)$

*Proof.* Let $s : \mathsf{J} \to \mathbb{R}^d$ and $\mathbf{s}_0 : \mathsf{I} \to \mathbb{R}^d$ be two continuously differentiable time seriess with $\mathsf{I} = (a, b), \mathsf{J} = (\alpha, \beta)$ two intervals of $\mathbb{R}$. By setting $\gamma : t \in \mathbb{R} \mapsto (\beta - \alpha)(t - a)/(b - a) + \alpha \in \mathbb{R}$, we have $\gamma(\mathsf{I}) = \mathsf{J}$ and $\gamma \in \mathcal{D}(\mathbb{R})$. By defining $f : (t, x) \in \mathbb{R}^{d+1} \mapsto x - \mathbf{s}_0(t) + s \circ \gamma(t)$, the map $\Phi_f \in \mathcal{D}(\mathbb{R}^{d+1})$, indeed, its inverse is $\Phi_f^{-1} : (t, x) \in \mathbb{R}^{d+1} \mapsto (t, x + \mathbf{s}_0(t) - s(t))$ and is continuously differentiable. Moreover, we have $\Pi_{\gamma, f}.\mathsf{G}(\mathbf{s}_0) = \{(\gamma(t), s \circ \gamma(t)) : t \in \mathsf{I}\} = \mathsf{G}(s)$.

Let $\bar{f} \in \mathrm{C}^1(\mathbb{R}^{d+1}, \mathbb{R}^d), \bar{\gamma} \in \mathcal{D}(\mathbb{R})$ and $\mathbf{s}_0 \in \mathrm{C}^1(\mathsf{I}, \mathbb{R}^d)$ with $\mathsf{I}$ an interval of $\mathbb{R}$. We have :

$$\begin{aligned}
\Pi_{\gamma, f}.\mathsf{G}(\mathbf{s}_0) &= \{(\gamma(t), f(t, \mathbf{s}_0(t))), \ t \in \mathsf{I}\} \\
&= \{(t, f\left(\gamma^{-1}(t), \mathbf{s}_0(\gamma^{-1}(t))\right), \ t \in \gamma(\mathsf{I})\} .
\end{aligned} \tag{10}$$

By defining $\bar{s} : t \in \gamma(\mathsf{I}) \to f\left(\gamma^{-1}(t), \mathbf{s}_0(\gamma^{-1}(t))\right)$, we have $\bar{s} \in \mathrm{C}^1(\gamma(\mathsf{I}), \mathbb{R}^d)$ by composition of $C^1$ functions and $\mathsf{G}(\bar{s}) = \Pi_{\gamma, f}.\mathsf{G}(\mathbf{s}_0)$ by (10), which concludes the proof. $\square$

**Lemma 2.** *If we denote by* $\mathsf{V}$ *the RKHS associated with the kernel* $K_\mathsf{G}$, *then for any vector field* $v$ *generated by* (5) *with* $v_0$ *satisfying* (4), *there exist* $\gamma \in \mathsf{D}(\mathbb{R})$ *and* $f \in \mathrm{C}^1(\mathbb{R}^{d+1}, \mathbb{R}^d)$ *such that* $\phi^v = \Psi_\gamma \circ \Phi_f$.

*Proof.* Let $v$ be a vector field generated by (5) with $v_0$ satisfying (4). We remark that the first coordinate of the velocity field $v_\tau$ denoted by $v_\tau^{\mathrm{time}}$ only depends on the time variable $t$ for any $\tau \in [0, 1]$. Thus, when computing the first coordinate of the deformation $\phi^v$, denoted by $\gamma$, we integrate (1) with $v_\tau$ replaced by $v_\tau^{\mathrm{time}}$, thus $\gamma$ is independant of the variable $x$. Moreover, $\gamma \in \mathcal{D}(\mathbb{R})$ since a Gaussian kernel induced an Hilbert space $\mathsf{V}$ satisfying $|f|_V \le |f|_\infty + |\,\mathrm{d}f|_\infty$ for any $f \in \mathsf{V}$ by [22, Theorem 9]. For the same reason, we have $\phi^v \in \mathcal{D}(\mathbb{R}^{d+1})$, and thus its last coordinates denoted by $f$ belongs to $\mathrm{C}^1(\mathbb{R}^{d+1}, \mathbb{R}^d)$, and by construction $\phi^v = \Psi_\gamma \circ \Phi_f$. $\square$

## C  Oriented varifold

In this section, we introduce the *oriented varifold* associated with curves. For further readings on curves and surfaces representation as varifolds, readers can refer to [30, 12]. We associate to

$\gamma \in C^1((a,b), \mathbb{R}^{d+1})$ an *oriented varifold* $\mu_\gamma$, i.e. a distribution on the space $\mathbb{R}^{d+1} \times \mathbb{S}^d$ defined as follows, for any smooth test function $\omega : \mathbb{R}^{d+1} \times \mathbb{S}^d \to \mathbb{R}$,

$$\mathbb{E}_{Y \sim \mu_\gamma}[\omega(Y)] = \mu_\gamma(\omega) = \int_a^b \omega\left(\gamma(t), \frac{\dot{\gamma}(t)}{|\dot{\gamma}(t)|}\right) |\dot{\gamma}(t)| \, dt \, .$$

Denoting by $W$ the space of smooth test function, we have that $\mu_\gamma$ belongs to its dual $W^*$. Thus, a distance on $W^*$ is sufficient to set a distance on oriented varifolds associated to curve and thus on $C^1((a,b), \mathbb{R}^{d+1})$ by the identification $\gamma \to \mu_\gamma$. Remark that in (TS-LDDMM), $\gamma$ should be the parametrization of a time series' graph $G(s)$, i.e. $\gamma : t \in I \to (t, s(t)) \in \mathbb{R}^{d+1}$ denoting by $s : I \to \mathbb{R}^d$ the time series. However, in practice, we work with discrete objects. That is why, we set $W$ as an RKHS to use its representation theorem. More specifically [30, Proposition 2 & 4] encourages us to consider a kernel $k : (\mathbb{R}^{d+1} \times \mathbb{S}^d)^2 \to \mathbb{R}$ such that there exist two positive and continuously differentiable kernels $k_{\text{pos}}$ and $k_{\text{dir}}$, such that for any $(x, \overrightarrow{u}), (y, \overrightarrow{v}) \in (\mathbb{R}^{d+1} \times \mathbb{S}^d)^2$

$$k((x, \overrightarrow{u}), (y, \overrightarrow{v})) = k_{\text{pos}}(x, y) k_{\text{dir}}(\overrightarrow{u}, \overrightarrow{v}) \, ,$$

with moreover $k_{\text{dir}} > 0$ and $k_{\text{pos}}$ which admits an RKHS $W_{\text{pos}}$ dense in the space of continous function on $\mathbb{R}^{d+1}$ vanishing at infinite [10].

Given such a kernel $k : (\mathbb{R}^{d+1} \times \mathbb{S}^d)^2 \to \mathbb{R}$ verifying [30, Proposition 2 & 4], we have that for any $(x, v) \in \mathbb{R}^{d+1} \times \mathbb{S}^d$, $\delta_{(x, \overrightarrow{v})}$ belongs to $W^*$ as a distribution and that the dual metric $\langle \cdot, \cdot \rangle_{W^*}$ satisfies for any $(x_1, v_1), (x_2, v_2) \in (\mathbb{R}^{d+1} \times \mathbb{S}^d)^2$,

$$\langle \delta_{(x_1, \overrightarrow{v}_1)}, \delta_{(x_2, \overrightarrow{v}_2)} \rangle_{W^*} = k((x_1, \overrightarrow{v}_1), (x_2, \overrightarrow{v}_2)) \, .$$

Thus, given two sets of triplets $X = (l_i, x_i, \overrightarrow{v}_i)_{i \in [T_0-1]} \in (\mathbb{R} \times \mathbb{R}^{d+1} \times \mathbb{S}^d)^{T_0-1}, Y = (l'_i, y_i, \overrightarrow{w}_i)_{i \in [T_1]} \in (\mathbb{R} \times \mathbb{R}^{d+1} \times \mathbb{S}^d)^{T_1-1}$ and denoting by

$$\mu_X = \sum_{i=1}^{T_0} l_i \delta_{(x_i, \overrightarrow{v}_i)}, \mu_Y = \sum_{i=1}^{T_1} l'_i \delta_{(y_i, \overrightarrow{w}_i)} \, , \tag{11}$$

we have,

$$\begin{aligned}
|\mu_X - \mu_Y|^2_{W^*} = & \sum_{i,j=1}^{T_0-1} l_i k((x_i, \overrightarrow{v_i}), (x_i, \overrightarrow{v_i}^0)) l_j + \sum_{i,j=1}^{T_1-1} l'_i k((y_i, \overrightarrow{w_i}), (y_i, \overrightarrow{w_i})) l'_j \\
& -2 \sum_{i=1}^{T_0-1} \sum_{j=1}^{T_1-1} l_i k((x_i, \overrightarrow{v_i}), (y_i, \overrightarrow{w_i})) l'_j \, .
\end{aligned}$$

Then, using the identification $X \to \mu_X, Y \to \mu_Y$, we can define a distance on sets of triplets as $d_{W^*,3}(X, Y) = |\mu_X - \mu_Y|^2_{W^*}$.

Now, we aim to discretize the oriented varifold $\mu_G$ related to a time series' graph $G(s)$ by using a set of triplets. This is carried out by using a discretized version of $G(s)$, i.e. $\tilde{G} = (g_i = (t_i, s(t_i)))_{i \in [T]} \in (\mathbb{R}^{d+1})^T$, in the following way: For any $i \in [T-1]$, denoting the center and length of the $i^{th}$ segment $[g_i, g_{i+1}]$ by $c_i = (g_i + g_{i+1})/2$, $l_i = \|g_{i+1} - g_i\|$, and the unit norm vector of direction $\overrightarrow{g_i g_{i+1}}$ by $\overrightarrow{v_i} = (g_{i+1} - g_i)/l_i$, we define the set of triplets $X(\tilde{G}) = (l_i, c_i, \overrightarrow{v_i})_{i \in [T-1]}$ and its related oriented varifold $\mu_{X(\tilde{G})} = \sum_{i=1}^{T-1} l_i \delta_{c_i, \overrightarrow{v_i}}$ as in (11). This is a valid discretization of the oriented varifold $\mu_G$ according to [30, Proposition 1]: $\mu_{X(\tilde{G})}$ converges towards $\mu_G$ as the size of the descretization mesh $\sup_{i \in [T-1]} |t_{i+1} - t_i|$ converges to 0.

Finally, we define a distance on discretized time series' graphs $\tilde{G}_1, \tilde{G}_2$ as $d_{W^*}(\tilde{G}_1, \tilde{G}_2) = d_{W^*,3}(X(\tilde{G}_1), X(\tilde{G}_2))$.

## C.1 Varifold kernels

Denote the one-dimensional Gaussian kernel by $K_\sigma^{(a)}(x, y) = \exp(-|x-y|^2/\sigma)$ for any $(x, y) \in (\mathbb{R}^a)^2$, $a \in \mathbb{N}$ and $\sigma > 0$. In the implementation, we use the following kernels, for any $((t_1, x_1), (t_2, x_2)) \in (\mathbb{R}^{d+1})^2, ((w_1, v_1), (w_2, v_2)) \in (\mathbb{S}^d)^2$,

$$k_{\text{pos}}(x, y) = K_{\sigma_{\text{pos},t}}^{(1)}(t_1, t_2) K_{\sigma_{\text{pos},x}}^{(d)}(x_1, x_2), \quad k_{\text{pos}}(x, y) = K_{\sigma_{\text{dir},t}}^{(1)}(w_1, w_2) K_{\sigma_{\text{dir},x}}^{(d)}(v_1, v_2) \, ,$$

where $\sigma_{\mathrm{pos},t}, \sigma_{\mathrm{pos},x}, \sigma_{\mathrm{dir},t}, \sigma_{\mathrm{dir},x} > 0$ are hyperparameters. In practice, we select $\sigma_{\mathrm{pos},x} \approx \sigma_{\mathrm{dir},x} \approx 1$ when the times series are centered and normalized. Otherwise we select $\sigma_{\mathrm{pos},x} \approx \sigma_{\mathrm{dir},x} \approx \bar{\sigma}_s$ with $\bar{\sigma}_s$ the average standard deviation of the time series. We choose $\sigma_{\mathrm{pos},t} \approx \sigma_{\mathrm{dir},t} = mf_e$ with $f_e$ the sampling frequency of the time series and $m \in [5]$ an integer depending on the time change between the starting and the target time series graph. The more significant the time change, the higher $m$ should be. The intuition comes from the fact that the width $\sigma_{\mathrm{pos},t}, \sigma_{\mathrm{dir},t}$ rules the time windows used to perform the comparison, and $\sigma_{\mathrm{pos},x}, \sigma_{\mathrm{dir},x}$ affects the space window. The size of the windows should be selected depending on the variations in the data.

## D  Tuning the hyperparameters of the TS-LDDMM velocity field kernel

The parameter $\sigma_{T,0}$ should be chosen *large* compared the sampling frequency $f_e$ and compared to average standard deviation $\bar{\sigma}_s$ of the time series, e.g $\sigma_{T,0} = 100$ as $\bar{\sigma}_s \approx f_e \approx 1$. It makes the time transformation smoother. If $\sigma_{T,0}$ is too small, for instance, $\sigma_{T,0} = f_e$, the effect of the time deformation is too localized, and there are not enough samples to make it visible.

The parameter $\sigma_{T,1}$ should be of the same order as $f_e$: two different points in time can have various space transformations. $\sigma_x$ should be of the same order of $\bar{\sigma}_s$: two points with a big difference regarding space compared to $\bar{\sigma}_s$ can have very different space transformations.

We take $c_0 \approx 10c_1$, we want to encourage time transformation before space transformation. We take $(c_0, c_1) = (1, 0.1)$ in all experiments.

## E  Experimental settings

All experiments were performed on a Debian 6.1.69-1 server with NVIDIA RTX A2000 12GB GPU, Intel(R) Xeon(R) Gold 5220R CPU @ 2.20GHz, and 250 GB of RAM. The source code is available on Github[5].

### E.1  Optimization details of TS-LDDMM & LDDMM

We implemented TS-LDDMM in Python with the JAX library [6].

**Initialization.**   As initialization of (8), all momentum parameters are set to $0$, and the initial graph of reference is picked from the dataset such that its length is equal to the median length observed in the dataset.

**Gradient descent.**   The chosen gradient descent method is "adabelief" [65] implemented in the OPTAX library [7]. The gradient descent has two main parameters: the number of steps (nb_steps) and the maximum stepsize value ($\eta_M$). The stepsize has a scheduling scheme:

- Warmup period on $0.1\times$ nb_steps steps: the stepsize increases linearly from $0$ to $\eta_M$. The goal is to learn progressively the parameters. If the step size is too large at the start, smaller steps at the end cannot make up for the mistakes made at the beginning.

- Fine tuning periode on $0.9\times$ nb_steps : the stepsize decreases from $\eta_M$ to $0$ with a cosine decay implemented in the OPTAX scheduler, i.e. the decreasing factor as the form $0.5(1 + \cos(\pi t/T))$.

By default, we set nb_steps to 400 and $\eta_M$ to 0.1.

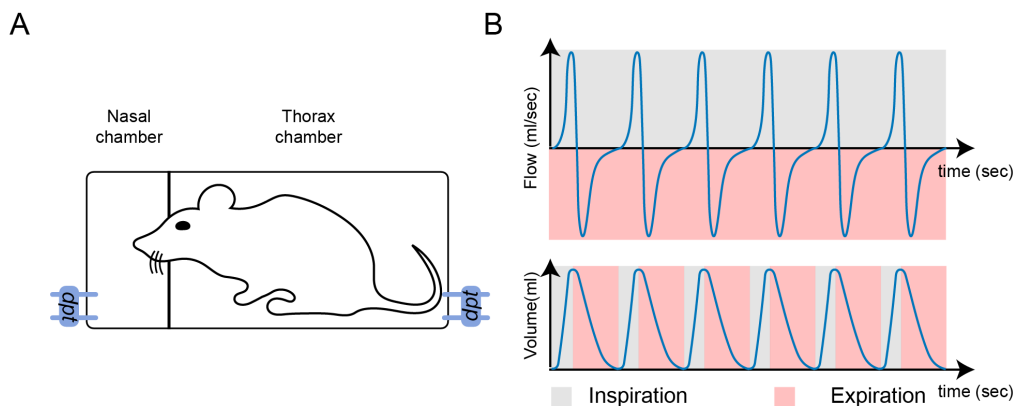

Figure 6: A: Illustration of a double-chamber plethysmograph. The term *dpt* stands for differential pressure transducer which measures the pressure in each compartment, the pressure then being converted to flow. B: Nasal airflow (top) and lung volume (bottom). During inspiration, airflow is positive (grey) and during expiration, airflow is negative (pink).

## F    Datasets

### F.1    Mouse respiratory cycle dataset

Ventilation is a simple physiological function that ensures a vital supply of oxygen and the elimination of $CO_2$. Acetylcholine (Ach) is a neurotransmitter that plays an important role in muscular activity, notably for breathing. Indeed, muscle contraction information passes from the brain to the muscle through the nervous system. Achs are located in synapses of the nervous system (central and peripheral) and skeletal muscles. They ensure the information transmission from nerve to nerve. However, the transmission cannot end without the hydrolysis of Ach by the enzyme Acetylcholinesterase (AchE), allowing nerves to return to their resting state. Inhibition of (AchE) with, for instance, nerve gas, pesticide, or drug intoxication leads to respiratory arrests.

The dataset comes from the experiment [41], where they studied the consequences of partial deficits in AChE and AChE inhibition on mice respiration. AchE inhibition was induced with an irritant molecule called physostigmine (an AchE inhibitor). Mice nasal airflows were sampled at 2000Hz with a Double Chamber plethysmograph [28], as depicted in Figure 6-A). The flow is expressed in $ml.s^{-1}$; it has a positive value during inspiration and a negative value expiration Figure 6-B). Among the mice population, we selected 7 control mice (**wt**) and 7 ColQ mice (**colq**), which do not have AChE anchoring in muscles and some tissues. As described in [41], mice experiments were as follows:

1. The mouse is placed in a DCP for 15 or 20 min to serve as an internal control.

2. The mouse is removed from the DCP and injected with physostigmine.

3. The mouse is placed back into the DCP, and its nasal flow is recorded for 35 or 40 min.

Respiratory cycles were extracted following procedure [21]. We removed respiratory cycles whose duration exceeds 1 second; the average respiratory cycle duration is 300 ms. We randomly sampled 10 respiratory cycles per minute and mouse. It leads to a dataset of 12,732 (time, genotype)-annotated respiratory cycles.

### F.2 Shape-based UCR/UEA time series classification datasets

We selected 15 shape-based datasets (7 univariates and 8 multivariates) from the from the University of East Anglia (UEA) and the University of California Riverside (UCR) Time Series Classification Repository[8] [15, 3]. All datasets were downloaded with the python package aeon[9]. Essential datasets information are summarized in Table 1 and further can be found in [15, 3].

Table 1: UCR/UEA shape-based time series datasets for classification.

|  | Dataset | Size | Lengh | Number of classes | Number of dimensions | Type |
|---|---|---|---|---|---|---|
| Univariate | ArrowHead | 211 | 251 | 3 | 1 | IMAGE |
|  | BME | 180 | 128 | 3 | 1 | SIMULATED |
|  | ECG200 | 200 | 96 | 2 | 1 | ECG |
|  | FacesUCR | 2250 | 131 | 14 | 1 | IMAGE |
|  | GunPoint | 200 | 150 | 2 | 1 | MOTION |
|  | PhalangesOutlinesCorrect | 2658 | 80 | 2 | 1 | IMAGE |
|  | Trace | 200 | 275 | 4 | 1 | SENSOR |
| Multivariate | ArticularyWordRecognition | 575 | 144 | 25 | 9 | SENSOR |
|  | Cricket | 180 | 1197 | 12 | 6 | MOTION |
|  | ERing | 60 | 65 | 6 | 4 | SENSOR |
|  | Handwriting | 1000 | 152 | 26 | 3 | MOTION |
|  | Libras | 360 | 45 | 15 | 2 | VIDEO |
|  | NATOPS | 360 | 51 | 6 | 24 | MOTION |
|  | RacketSports | 303 | 30 | 4 | 6 | SENSOR |
|  | UWaveGestureLibrary | 240 | 315 | 8 | 3 | SENSOR |

## G   Appendix for experiment: TS-LDDMM representation identifiability

In this experiment, we evaluate the ability of TS-LDDMM to retrieve the parameter $v_0^*$ that encodes the deformation $\varphi^{\{v_0^*\}}$ acting on a time series graph $\mathsf{G}$ by solving the geodesic shooting problem (6) between $\mathsf{G}$ and $\varphi^{\{v_0^*\}}.\mathsf{G}$. Parameter identifiability is an important property for subsequent statistical analysis. Results show that TS-LDDMM representations are identifiable or weakly identifiable depending on the velocity field kernel $K_G$ specification.

### G.1   Settings

This experiment only involves the TS-LDDMM method in two different settings:

- **The velocity field kernel $K_G$ is well-specified:** The velocity field kernel $K_G$ is set to $(c_0, c_1, \sigma_{T,0}, \sigma_{T,1}, \sigma_x) = (1, 0.1, 100, 1, 1)$, the varifold loss kernels $(k_{pos}, k_{dir})$ are set to $(\sigma_{\mathrm{pos},t}, \sigma_{\mathrm{pos},t}, \sigma_{\mathrm{dir},t}, \sigma_{\mathrm{dir},x}) = (2, 1, 2, 0.6)$, and the optimizer has 400 steps with a maximum stepsize $\eta_M$ of 0.05.

- **The velocity field kernel $K_G$ is missspecified:** The velocity field kernel $K_G$ is set with $(c_0, c_1, \sigma_{T,1}) = (1, 0.1, 1)$, $\sigma_{T,0}$ ranging in $(1, 5, 10, 50, 100, 200, 300)$, and $\sigma_x$ ranging in $(0.1, 1, 10, 100)$. The varifold loss kernels $(k_{pos}, k_{dir})$ are set to $(\sigma_{\mathrm{pos},t}, \sigma_{\mathrm{pos},t}, \sigma_{\mathrm{dir},t}, \sigma_{\mathrm{dir},x}) = (2, 1, 2, 0.6)$, and the optimizer has 400 steps with a maximum stepsize $\eta_M$ of 0.05.

provided that the hyperparameters and the reference graph are wisely selected, i.e., the parameter $v_0^*$ generating a deformation $\varphi^{\{v_0^*\}}$ of a time series graph $\mathsf{G}$ can be estimated from the data $\mathsf{G}$, $\varphi^{\{v_0^*\}}.\mathsf{G}$ by solving the geodesic shooting problem (6).

**The velocity field kernel $K_G$ is well specified.**   First, we show the model identifiability when the kernel $K_G$ is well specified: the estimated parameter is a good approximation of the generating parameter when the generation and the estimation procedure use the same hyperparameters for the RKHS kernel $K_\mathsf{G}$. All the hyperparameter values for generation and estimation are given in Appendix G.1.

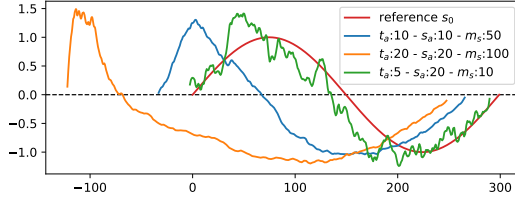

Figure 7: Plots of $\varphi^{\{v_0(\alpha^*, \mathsf{X})\}}.\mathsf{X}$ for different values of $\alpha^*$ according to its sampling parameter $t_a, s_a, m_s$, taking $\mathsf{X} = \mathsf{G}(s_0)$ with $s_0 : k \in [300] \to \sin(2\pi k/300)$.

Table 2: Values of $\mathscr{L}(\varphi^{\{v_0(\alpha^*, \mathsf{X})\}}.\mathsf{X}, \varphi^{\{\hat{v}_0\}}.\mathsf{X})$ as $\alpha^*$ is sampled according to Gen(10,10,50) and $\hat{v}_0$ is estimated using $K_\mathsf{G}$ with varying parameters $\sigma_{T,1}, \sigma_x$.

| $\sigma_{T,0} \backslash \sigma_x$ | 1 | 10 | 50 | 100 | 200 | 300 |
|---|---|---|---|---|---|---|
| 0.1 | 2e+0 | 3e-4 | 1e-5 | 4e-6 | 7e-4 | 4e-3 |
| 1 | 4e-2 | 1e-4 | 1e-5 | 4e-6 | 7e-4 | 4e-3 |
| 100 | 4e-2 | 2e-4 | 1e-5 | 4e-6 | 7e-4 | 4e-3 |

We fix the initial control points as $\mathsf{X} = (x_k = (k, \sin(2\pi k/300)))_{k \in [300]}$. Given $m_s \in \mathbb{N}_{>0}$ and $t_a, s_a > 0$, we randomly generate initial momentums $\alpha^* = (\alpha_k^*)_{k \in [\mathbf{n}_0]}$ with the following sampling, called Gen$(m_s, t_a, s_a)$: For any $k \in [\mathbf{n}_0]$, $\alpha_k'$ is sampled according to a Gaussian normal distribution $\mathcal{N}(0_{d+1}, I_{d+1})$. Then, $(\alpha_k')_{k \in [\mathbf{n}_0]}$ is regularized by a rolling average of size $m_s$, we get $\bar{\alpha}' = (\bar{\alpha}_k')_{k \in [\mathbf{n}_0]}$. Finally, we normalize $\bar{\alpha}'$ to derive $\alpha^*$ such that $|([\alpha_k^*]_t)_{k \in [\mathbf{n}_0]}| = t_{\text{amp}}$ and $|([\alpha_k^*]_s)_{k \in [\mathbf{n}_0]}| = s_{\text{amp}}$ for any $k \in [\mathbf{n}_0]$, denoting by $[\alpha_k^*]_t, [\alpha_k^*]_s$ the time and space coordinates of $\alpha_k^*$ respectively. Note that the regularizing step $(\alpha_k')_{k \in [\mathbf{n}_0]} \to \bar{\alpha}'$ is necessary to obtain realistic deformations which take into account the regularity induced by the RKHS $\mathsf{V}$.

Then, using $v_0(\alpha^*, \mathsf{X})$ as defined in (4) with initial momentums $\alpha^*$ and control points $\mathsf{X}$, we apply the induced deformation $\varphi^{\{v_0\}}$ by (5) to $\mathsf{X}$ and obtain $\varphi^{\{v_0\}}.\mathsf{X}$. Finally, we solve (6) to recover an estimation $\hat{\alpha}$ of $\alpha^*$ and report the average relative error (ARE) $|v_0(\hat{\alpha}, \mathsf{X}) - v_0(\alpha^*, \mathsf{X})|_\mathsf{V}/|v_0(\alpha^*, \mathsf{X})|_\mathsf{V}$ on 50 repetitions. This procedure is performed for any $m_s, t_a, s_a \in \{10, 50, 100\} \times \{5, 10, 15, 20\}^2$. Mean, standard deviation, and maximum of the ARE on all these hyperparameters choices are respectively **0.10**, **0.03**, **0.17**. Therefore, the estimation procedure (6) offers a good approximation of the true parameter when the kernel $K_\mathsf{G}$ is well specified. We observe that the estimation is difficult when $t_a \ll s_a$ because the time series can be very noisy as illustrated in Figure 7: this impacts the Varifold loss which is sensitive to tangents.

**The velocity field kernel $K_G$ is misspecified.** We demonstrate a weak identifiability when the kernel $K_\mathsf{G}$ is misspecified: we can reconstruct the graph time series' after deformations even if the hyperparameters of $K_\mathsf{G}$ are different during the generation and the estimation. The hyperparameters of $K_\mathsf{G}$ during generation are $(c_0, c_1, \sigma_{T,0}, \sigma_{T,1}, \sigma_x) = (1, 0.1, 100, 1, 1)$ and we fix $\sigma_{T,1}, c_0, c_1 = (1, 1, 0.1)$ for $K_\mathsf{G}$ during estimation. We aim to understand the impact of $\sigma_{T,1}, \sigma_x$ on the reconstruction since they are encoding the smoothness of the transformation according to time and space.

For any choice of the hyperparameters $\sigma_{T,1}, \sigma_x \in \{1, 10, 50, 100, 200, 300\} \times \{0.1, 1, 100\}$ related to $K_\mathsf{G}$ in the estimation, we average $\mathscr{L}(\varphi^{\{v_0(\alpha^*, \mathsf{X})\}}.\mathsf{X}, \varphi^{\{\hat{v}_0\}}.\mathsf{X})$ on 50 repetitions when $\alpha^*$ is sampled according to Gen$(10, 10, 50)$ and $\hat{v}_0 = v_0(\hat{\alpha}, \mathsf{X})$ denoting by $\hat{\alpha}$ the result of the minimization (6). We observe in Table 2 that the reconstruction is almost perfect except in the case when $\sigma_{t,0} = 1$ during estimation, while $\sigma_{t,0} = 100$ during generation. Compared to $\sigma_{T,0}$, $\sigma_x$ has nearly no impact on the reconstruction. In Appendix C.1-D, we propose guidelines to drive future hyperparameters tuning and further discussions related to $\sigma_{T,1}, c_0, c_1$.

# H Appendix for experiment: Robustness to irregular sampling

This experiment is inspired by [43] where the authors perform an extensive comparison of Neural Ordinary Differential Equations (Neural ODEs) methods [31]. We assess the classification performances of several methods under regular sampling (0% missing rate) and three irregular sampling regimes on 15 shape-based datasets (7 univariate & 8 multivariate). Methods and training strategy are taken from its associated Github[10] and described in what follows. We conclude with the results, which show that our method, TS-LDDMM, outperforms all methods for sampling regimes with missing rates: 0%, 30%, and 50%.

## H.1 Benchmark methods

In related work, we give an overview of Neurals ODEs methods and their relation with TS-LDDMM.

- RNN-based methods: Baseline reccurent neural networks including RNN [38], LSTM [27], and GRU [14].
- Attention-based methods: Multi-Time Attention Networks (MTAN) [52] and Multi-Integration Attention Module (MIAM) [35]. Both handle multivariate time series irregularly sampled with attention mechanisms.
- Neural ODEs: ODE-LSTM [34] a form of Neural-ODEs used to learn continuous latent representations.
- Neural SDEs: Neural SDE [36] and Neural LNSDE [43] have been proposed to model randomness in time-series using drift and diffusion terms as an extension of Neural-ODEs.
- Shape-Analysis methods: TS-LDDMM (ours) and LDDMM [23]. From shape analysis, both methods learn representations by solving ODEs parametrized with Kernels. While both methods handle multivariate signals irregularly sampled, TS-LDDMM is specifically designed for time series.

## H.2 Model architecture

**Neural ODEs methods**   As depicted in [43], any Neural ODEs layer in Appendix H.1 is followed by an MLP with two fully connected layers with `ReLU` activations. The risk of overfitting and the model regularization are handled with a dropout rate of 10% and an early-stopping mechanism, ceasing the training when the validation loss does not improve for 10 successive epochs.

For each method and dataset, the learning rate, the hidden vector dimensions, and the number of layers are optimized to minimize the CrossEntropy loss on a validation set using the `Ray` [11] Python library. The learning rate varies from $10^{-4}$ to $10^{-1}$ using log uniform search, the hidden vector dimension ranges from $16, 32, 64, 128$ using grid search, and the number of layers ranges from $1, 2, 3, 4$ using grid search. The batch size was selected from $16, 32, 64, 128$ according to the size of the dataset. All methods were trained for 100 epochs, and the best method was selected based on the lowest validation loss.

**TS-LDDMM and LDDMM**   Representations learned with TS-LDDMM or LDDMM are fed to a Support Vector Classifier (SVC) from `scikit-learn` [12]. All SVC's hyperparameters are set to default except the regularization term C, which is set through grid search on a validation set with the macro f1-score [13].

To learn TS-LDDMM (resp. LDDMM) representations, the velocity field kernel $K_G$ is set to $(c_0, c_1, \sigma_{T,0}, \sigma_{T,1}, \sigma_x) = (1, 0.1, 0.33\bar{l}, 1, n_d)$, (resp. $(\sigma_T, \sigma_x) = (0.33\bar{l}, n_d)$) where $\bar{l}$ is the average time series length and $n_d$ the number of dimensions. For both methods and all datasets, the varifold loss kernels $(k_{pos}, k_{dir})$ are identical and set to $(\sigma_{\mathrm{pos},t}, \sigma_{\mathrm{pos},t}, \sigma_{\mathrm{dir},t}, \sigma_{\mathrm{dir},x}) = (2, n_d, 2, n_d)$. For TS-LDDMM (resp. LDDMM), the optimizer is set with 400 epochs (resp. 400) and a maximum learning rate $\eta_M = 0.1$ (resp. $\eta_M = 0.01$). In all cases, the initial reference graph is selected in the dataset as a time series with the median length.

### H.3 Protocol

In this experiment, we investigate the robustness to missing samples and the classification performance of TS-LDDMM compared to Neural ODEs on 15 datasets described in Appendix F.2. For fairness between methods of different architectures, the evaluation protocol on each dataset and method is as follows:

1. Spilt the dataset in train 75%, validation 15%, and test 15%.

2. Tune hyperparameters with train and validation sets and a missing rate of 0%.

3. For each missing rate in [0%,30%,50%,70%]
   - Remove samples in time series in the train and test sets according to the missing rate and the drop procedure described in [31].
   - Train the model on the train set
   - Evaluate the macro f1-score on the test set

### H.4 Results

In this experiment, we investigate the robstuness to missing samples and the classification performance of TS-LDDMM representations. We compare TS-LDDMM with LDDMM and 8 neural ODEs networks. Performances are evaluated in terms of average macro f1-score and rank on four different regimes of missing rate 0%,30%,50%, and 70%. Results are aggregated in Table 3.

On three out of four regimes (0%,30%, and 50%) TS-LDDMM classifier is the best performer in terms of f1-score and rank. For missing rates of 0% and 30%, the score increases by 10% compared to the second-best performer, LDDMM. However, LDDMM is not the second-best performer in rank (Neural LNSDE), showing its sensitivity to parameterization, unlike TS-LDDMM, which remains consistent. Performances of Neural LNSDE remain constant with the increase of the missing rate as observed in [43], and it becomes the best performer for missing rate 70%. The decrease in TS-LDDMM performances with the increasing missing rate is due to the varifold loss, which poorly approximates the time series shape. Other losses might be more relevant for high missing rates.

Overall, TS-LDDMM is a relevant and consistent shape-based representation for irregularly sampled multivariate time series for missing rates up to 50% .

Table 3: Comparison of average macro f1-score and rank as the sample dropping rate increases. **First** & second best performers. TS-LDDMM is the best performer on three out of four regimes.

| Methods | Regular | | 30 % dropped | | 50 % dropped | | 70 % dropped | |
|---|---|---|---|---|---|---|---|---|
| | F1-score | Rank | F1-score | Rank | F1-score | Rank | F1-score | Rank |
| RNN (1999) | $0.64 \pm 0.21$ | 6.2 | $0.53 \pm 0.23$ | 6.6 | $0.48 \pm 0.21$ | 7.2 | $0.44 \pm 0.21$ | 6.07 |
| LSTM (1997) | $0.61 \pm 0.29$ | 6.0 | $0.57 \pm 0.29$ | 6.27 | $0.53 \pm 0.25$ | 6.07 | $0.51 \pm 0.29$ | 5.27 |
| GRU (2014) | $0.71 \pm 0.26$ | 4.2 | $0.68 \pm 0.28$ | 4.27 | $0.66 \pm 0.28$ | 3.73 | $0.59 \pm 0.28$ | 3.67 |
| MTAN (2021) | $0.59 \pm 0.28$ | 7.13 | $0.58 \pm 0.28$ | 5.8 | $0.54 \pm 0.29$ | 5.33 | $0.51 \pm 0.28$ | 5.0 |
| MIAM (2022) | $0.48 \pm 0.35$ | 6.93 | $0.42 \pm 0.33$ | 8.27 | $0.47 \pm 0.31$ | 6.93 | $0.35 \pm 0.31$ | 7.6 |
| ODE-LSTM (2020) | $0.63 \pm 0.24$ | 6.0 | $0.57 \pm 0.25$ | 6.53 | $0.51 \pm 0.24$ | 7.27 | $0.45 \pm 0.23$ | 6.73 |
| Neural SDE (2019) | $0.48 \pm 0.28$ | 7.67 | $0.47 \pm 0.26$ | 7.47 | $0.45 \pm 0.27$ | 7.13 | $0.45 \pm 0.25$ | 6.0 |
| Neural LNSDE (2024) | $0.7 \pm 0.27$ | 3.87 | $0.68 \pm 0.29$ | 4.0 | $0.67 \pm 0.25$ | 3.53 | **$0.66 \pm 0.23$** | **2.47** |
| LDDMM (2008) | $0.72 \pm 0.2$ | 4.53 | $0.7 \pm 0.21$ | 4.2 | $0.57 \pm 0.25$ | 5.0 | $0.4 \pm 0.25$ | 7.13 |
| TS-LDDMM (ours) | **$0.83 \pm 0.18$** | **2.93** | **$0.8 \pm 0.18$** | **2.07** | **$0.7 \pm 0.26$** | **3.33** | $0.51 \pm 0.27$ | 5.67 |

## I  Appendix for experiment: Classification benchmark on regularly sampled datasets

In this section, we compare the classification performances of TS-LDDMM with other methods from shape analysis on 15 shape-based datasets of time series regularly sampled. TS-LDDMM outperforms other methods on 12 out of 15, highlighting its relevance for shape analysis when dealing with time series.

## I.1 Benchmark methods

- SRV-based method: we include TCLR [26] a logistic regression on the tangent space of the Frechet mean with Square Root Velocity (SRV representation). We also include Shape-FPCA [60] that encodes both the time series and its time parameterization.

- LDDMM-Based : TS-LDDMM (ours) and LDDMM [23]. Both methods learn representations by solving ODEs parametrized with Kernels. While both methods handle multivariate signals, TS-LDDMM is specifically designed for time series.

## I.2 Model settings

**TCLR & Shape-FPCA**   Shape-FPCA is available in the Python library `FDASRSF` [14]. Once the shape-FPCA representations are learned, they are fed to an SVC from `scikit-learn`. FDASRSF provides SRV representation methods that we combined with a logistic regression from `scikit-learn` to implement TCLR. For both methods, the number of steps to learn the Frechet mean is set to 50, and the regularization hyperparameter C is set through grid search on a validation set with the macro f1-score. Other parameters are set to default.

**TS-LDDMM & LDDMM**   Representations learned with TS-LDDMM or LDDMM are fed to an SVC from `scikit-learn`. All SVC's hyperparameters are set to default except the regularization term C, which is set through grid search on a validation set with the macro f1-score.

To learn TS-LDDMM (resp. LDDMM) representations, the velocity field kernel $K_G$ is set to $(c_0, c_1, \sigma_{T,0}, \sigma_{T,1}, \sigma_x) = (1, 0.1, 0.33\bar{l}, 1, n_d)$, (resp. $(\sigma_T, \sigma_x) = (0.33\bar{l}, n_d)$) where $\bar{l}$ is the average time series length and $n_d$ the number of dimensions. For both methods and all datasets, the varifold loss kernels $(k_{pos}, k_{dir})$ are identical and set to $(\sigma_{\text{pos},t}, \sigma_{\text{pos},t}, \sigma_{\text{dir},t}, \sigma_{\text{dir},x}) = (2, n_d, 2, n_d)$. For TS-LDDMM (resp. LDDMM), the optimizer is set with 400 epochs (resp. 400) and a maximum learning rate $\eta_M = 0.1$ (resp. $\eta_M = 0.01$). In all cases, the initial reference graph is selected in the dataset as a time series with the median length.

## I.3 Protocol

For each dataset and method, the evaluation protocol is a simple train,validation test with hyperparameter tuning:

1. Split The dataset in train 75%, validation 15%, and test 15%.
2. Training and hyperparameters tuning with train and validation sets
3. Evaluate the macro f1-score on the test set

## I.4 Results

In this experiment, we investigate the classification performances of several methods from shape analysis on 15 shape-based time series datasets (7 univariate and 8 multivariate). The performances are evaluated in terms of macro f1-score. Results are aggregated in Table 4.

The TS-LDDMM-based classifier outperforms other methods on 12 out of 15 datasets. TCLR is the second-best performer on univariate datasets; however, its current implementation with `FDASRSF` does not extend to the multivariate case, which limits usage. LDDMM performances are lower than TCLR, and Shape-FPCA is the worst performer.

Overall, TS-LDDMM representations are well suited for shape-based time series classification, and its extension to multivariate irregularly sampled time series makes it a relevant option for time series shape analysis.

## J   Appendix for the experiment: Noise sensitivity

This experiment evaluates the influence of noise on the learning of the reference sequence for TS-LDDMM and SRVF Kacher-mean, a subroutine of Shape-FPCA [60].

Table 4: F1-score comparison between methods from shape analysis on 15 datasets. **First** and second best performers.

| | Dataset | Shape-FPCA (2024) | TCLR (2024) | LDDMM (2008) | TS-LDDMM (ours) |
|---|---|---|---|---|---|
| | ArrowHead | 0.18 | 0.75 | 0.84 | **0.91** |
| | BME | 0.16 | 1.00 | 0.82 | **1.00** |
| | ECG200 | 0.40 | 0.67 | **0.81** | 0.79 |
| Univariate | FacesUCR | 0.08 | 0.73 | 0.69 | **0.86** |
| | GunPoint | 0.93 | 0.97 | 0.83 | **1.00** |
| | PhalangesOutlinesCorrect | 0.39 | **0.63** | 0.53 | 0.52 |
| | Trace | 0.55 | 1.00 | 0.46 | **1.00** |
| | ArticularyWordRecognition | – | – | 0.98 | **1.00** |
| | Cricket | – | – | 0.77 | **0.93** |
| | ERing | – | – | 0.95 | **0.98** |
| Multivariate | Handwriting | – | – | 0.22 | **0.44** |
| | Libras | – | – | 0.56 | **0.60** |
| | NATOPS | – | – | 0.82 | **0.82** |
| | RacketSports | – | – | **0.83** | 0.79 |
| | UWaveGestureLibrary | – | – | 0.72 | **0.81** |

## J.1 Protocol

The dataset includes 100 sine waves with randomly generated time parametrization by following the procedure described in Appendix G.1 with $\mathrm{Gen}(50, 1, 0)$ and uniformly resampled. The dataset has been altered under four scenarios with an additive Gaussian noise centered and with standard deviation $\sigma_\epsilon \in \{0, 0.05, 0.1, 0.2\}$. The referent sequence is learned for each scenario, and the $L_2$-norm error between the exact and the learned barycenter is computed.

## J.2 Method settings

For SRVF, the number of steps to learn the Kacher-mean is set to 20. Regarding TS-LDDMM, the velocity field kernel $K_G$ is set to $(c_0, c_1, \sigma_{T,0}, \sigma_{T,1}, \sigma_x) = (1, 0.1, 65, 1, 1)$, and the varifold loss kernels $(k_{pos}, k_{dir})$ are set to $(\sigma_{\mathrm{pos},t}, \sigma_{\mathrm{pos},t}, \sigma_{\mathrm{dir},t}, \sigma_{\mathrm{dir},x}) = (5, 1, 1, 1)$. For TS-LDDMM, the optimizer is set with 400 epochs and a maximum learning rate $\eta_M = 0.1$. In all cases, the initial reference graph is selected in the dataset as a time series with the median length.

## J.3 Results

Figure 8 illustrates the results. Noise level affects the learning reference graph in both cases, as depicted by the increasing error and the illustrations. However, the overall sine wave shape is better preserved by TS-LDDMM compared to SRVF Kacher-mean, for which the sine wave amplitude decreases as the noise increases. In addition, for TS-LDDMM, the regularity of the reference graph can be controlled by penalizing the norm of the velocity fields in the loss function. Further work on penalization will be conducted to handle noisy data better.

# K    Appendix for experiment: Analysis of respiratory behavior in mice

## K.1    Settings

This experiment involves TS-LDDMM, LDDMM [23] and Shape-FPCA [60] methods. Two scenarios are investigated: before drug exposure and before/after drug exposure. All methods are investigated on both scenarios.

**TS-LDDMM parameters.**

- **Before exposure:** The velocity field kernel $K_G$ is set to $(c_0, c_1, \sigma_{T,0}, \sigma_{T,1}, \sigma_x) = (1, 0.1, 150, 1, 2)$. The varifold loss is the sum of three varifolds to capture shapes variations at different scales with parameters: (Varifold 1,Varifold 2,Varifold 3): $((5, 2, 5, 1), (2, 1, 2, 0.6), (1, 0.6, 1, 0.6))$ and the mapper $(\sigma_{\mathrm{pos},t}, \sigma_{\mathrm{pos},t}, \sigma_{\mathrm{dir},t}, \sigma_{\mathrm{dir},x})$. The optimizer has 800 steps with a maximum stepsize $\eta_M$ of 0.3.
- **Before/after exposure:** The velocity field kernel $K_G$ is set to $(c_0, c_1, \sigma_{T,0}, \sigma_{T,1}, \sigma_x) = (1, 0.1, 220, 1, 2)$. The varifold loss is the sum of four varifolds to capture shapes

## (a) TS-LDDMM barycenter

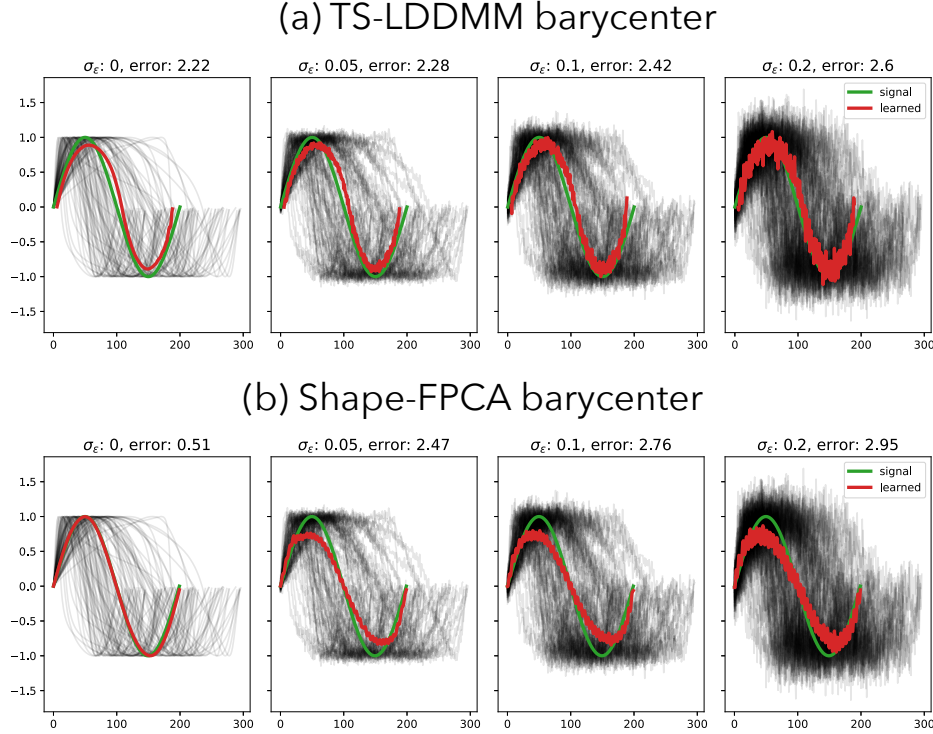

## (b) Shape-FPCA barycenter

Figure 8: Illustration of the learned barycenter (red) compared to the exact barycenter (green) for both TS-LDDMM **(a)** and Shape-FPCA **(b)**. The computation has been done for different level of noise $\epsilon \sim \mathcal{N}(0, \sigma_\epsilon)$ with $\sigma_\epsilon \in \{0, 0.05, 0.1, 0.2\}$.

variations at different scales with parameters: (Varifold 1,Varifold 2,Varifold 3, Varifold 4): $((30, 2, 30, 1), (5, 2, 5, 1), (2, 1, 2, 0.6), (1, 0.1, 1, 0.1))$ and the mapper $(\sigma_{\text{pos},t}, \sigma_{\text{pos},t}, \sigma_{\text{dir},t}, \sigma_{\text{dir},x})$. The optimizer has 800 steps with a maximum stepsize $\eta_M$ of 0.3.

**LDDMM parameters.** Note that varifold losses are unchanged between TS-LDDMM and LD-DMM. Compared to TS-LDDMM, the convergence of LDDMM is more sensitive to the maximum stepsize $\eta_m$, which must remain small for LDDMM to guarantee the convergence.

- **Before exposure:** The velocity field kernel $K_G$ is an anysotropic Gaussian kernel with parameters $\sigma_T = 150$ for the time dimension and $\sigma_x = 2$ for space dimensions. The varifold loss is the sum of three varifolds to capture shapes variations at different scales with parameters: (Varifold 1,Varifold 2,Varifold 3): $((5, 2, 5, 1), (2, 1, 2, 0.6), (1, 0.6, 1, 0.6))$ and the mapper $(\sigma_{\text{pos},t}, \sigma_{\text{pos},t}, \sigma_{\text{dir},t}, \sigma_{\text{dir},x})$. The optimizer has 800 steps with a maximum stepsize $\eta_M$ of 0.01.

- **Before/after exposure:** The velocity field kernel $K_G$ is an anysotropic Gaussian kernel with parameters $\sigma_T = 220$ for the time dimension and $\sigma_x = 2$ for space dimensions. The varifold loss is the sum of four varifolds to capture shapes variations at different scales with parameters: (Varifold 1,Varifold 2,Varifold 3, Varifold 4): $((30, 2, 30, 1), (5, 2, 5, 1), (2, 1, 2, 0.6), (1, 0.1, 1, 0.1))$ and the mapper $(\sigma_{\text{pos},t}, \sigma_{\text{pos},t}, \sigma_{\text{dir},t}, \sigma_{\text{dir},x})$. The optimizer has 800 steps with a maximum stepsize $\eta_M$ of 0.01.

**Shape-FPCA parameters.** For both scenarios, respiratory cycles are linearly interpolated and resampled to 200 points, and the length of the original time interval is kept. The computation of the Kacher-mean is done in a maximum of 50 iterations, and srv representations of the realigned time series and time parametrization are concatenated with cycle durations. When concatenating these vectors, the choice of amplitude factors is made to minimize the reconstruction error from the

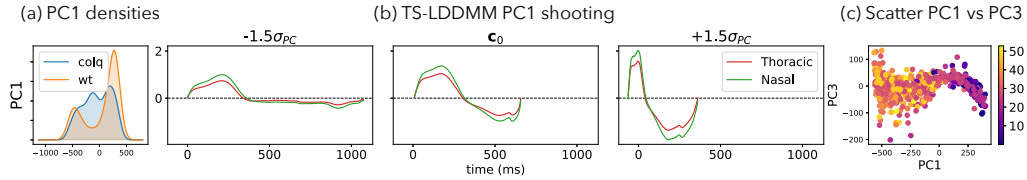

Figure 9: Analysis of the first Principal Component (PC1) related to mice ventilation before and after exposure with TS-LDDMM representations. **(a)** displays PC densities per mice genotype, **(b)** illustrates deformations of the reference respiratory cycle $c_0$ along PC1, and **(c)** displays all respiratory cycles with respect to time in PC1 and PC3 coordinates

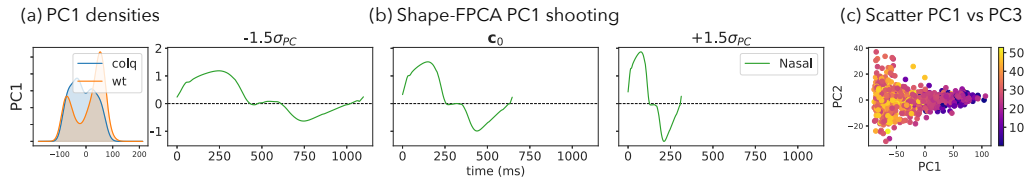

Figure 10: Analysis of the first Principal Component (PC1) related to mice ventilation before and after exposure with shape-FPCA representations. **(a)** displays PC densities per mice genotype, **(b)** illustrates deformations of the reference respiratory cycle $c_0$ along PC1, and **(c)** displays all respiratory cycles with respect to time in PC1 and PC2 coordinates

principal components analysis by following the procedure described in [60]. Shape-FPCA does not handle multivariate data, and we only kept the nasal airflow for this method.

## K.2    Addiotinal results

Figure 9 presents results for TS-LDDMM and Figure 10 presents results for Shape-FPCA. The main components look similar. However, a subtle difference, yet important, can be noticed. With Shape-FPCA, the deformation tends to be a uniform time scaling, whereas, with TS-LDDMM, the time dilatation mainly occurs during the pause between inspiration and expiration. Qualitatively, this last deformation fits the physiological phenomenon: Mice's muscles cannot relax after exposure to the irritant molecule, leading to pauses between inspiration and expiration [41]. Qualitatively, contrary to Shape-FPCA, which manages to represent the main phenomena in the data, the deformations of TS-LDDMM capture subtle physiological behaviors essential for understanding the phenomenon at hand.

